# SS1: Accelerating Inference with Fast and Expressive Sketch Structured Transform

**Kimia Saedi**[1] *   **Aditya Desai**[1] *   **Apoorv Walia**[1]   **Jihyeong Lee**[2] †
**Keren Zhou**[2]   **Anshumali Shrivastava**[1,3]
[1]Rice University   [2]George Mason University
[3]Ken Kennedy Institute, ThirdAI Corp., Xmad.ai
{ks152,apd10,aw82,as143}@rice.edu
{jlee436,kzhou6}@gmu.edu

## Abstract

Tensor multiplication with learned weight matrices is the fundamental building block in deep learning models. These matrices can often be sparsified, decomposed, quantized, or subjected to random parameter sharing without losing accuracy, suggesting the possibility of more efficient transforms. Although many variants of weight matrices exist, unstructured ones are incompatible with modern hardware, slowing inference and training. On the other hand, structured variants often limit expressivity or fail to deliver the promised latency benefits. We present Sketch Structured Transform(SS1), an expressive and GPU-friendly operator that accelerates inference. SS1 leverages parameter sharing in a random yet structured manner to reduce computation while retraining the rich expressive nature of parameter sharing. We confirm empirically that SS1 offers better quality-efficiency tradeoffs than competing variants. Interestingly SS1 can be combined with Quantization to achieve gains unattainable by either method alone, a finding we justify via theoretical analysis. The analysis may be of independent interest. Moreover, existing pre-trained models can be projected onto SS1 and finetuned for efficient deployment. Surprisingly, these projected models can perform reasonably well even without finetuning. Our experiments highlight various applications of the SS1: (a) Training GPT2 and DLRM models from scratch for faster inference. (b) Finetuning projected BERT models for 1.31× faster inference while maintaining GLUE scores. (c) Proof of concept with Llama-3-8b, showing 1.11× faster wall clock inference using projected SS1 layers without finetuning. Our code is open-source.[3]

## 1   Introduction

Tensor-matrix multiplication is one of the fundamental operations in deep learning models across various domains. Linear transformation are especially crucial in transformer architectures, which form the backbone of foundational models responsible for the advanced capabilities of Large Language Models (LLMs). A significant portion of the computational load in LLMs comes from linear layers. For example, with a batch size of 4 at full sequence length, the MLP workload of even the smallest Llama-3-8B model [1] involves ($32768 \times 4096 \times 4096$) operations in attention and ($32768 \times 4096 \times 14336$) in the MLP. Larger models have even larger workloads. Given these demands, finding efficient and expressive alternatives to standard linear layers is a crucial research direction.

Deep learning models often have significant redundancies that can be removed using various techniques. Popular approaches include Sparsification [2, 3, 4, 5, 6], Quantization [2, 7], Randomized

Parameter Sharing (RPS) [8, 9, 10], and Low-rank decomposition [11, 3]. Unstructured sparsity, while expressive, is inefficient on modern hardware. Structured sparsity [12, 13, 14] and low-rank are efficient but often compromise quality. Recently, it was shown in [15] that sparsity is theoretically weaker than RPS methods regarding the quality of learned models under compression. However, traditional RPS methods only focus on reducing the parameter memory footprint and do not affect FLOPs or latency. This paper proposes Sketch Structured Transform (SS1), an RPS scheme that reduces FLOPs while maintaining quality and improving wall-clock inference for machine learning models.

Randomized parameter sharing (RPS) reduces the model's memory footprint by randomly tying parameters across the model. While these methods offer a superior memory-quality tradeoff, they do not alter the computational graph, resulting in similar FLOPs and latency. Addressing this issue – reducing computation and latency while retaining the memory-quality tradeoff – has been a key challenge with RPS methods. In this paper, we propose a random yet structured parameter-sharing method that maintains superior quality, reduces the number of FLOPs, and improves the wall-clock latency of linear layers which are compute bound in most workloads.

The key idea in SS1 is to tie parameters inside a single neuron weight. This weight-tying can be equivalently implemented as the input being reduced in dimension first and then multiplied with the compressed weight vector which reduces FLOPs and memory movement in a neuron computation. We show how to devise this tying in a GPU-friendly manner using what we call $K-$ and $N-$ coalescing. Additionally, SS1 can be integrated with existing RPS methods to obtain independent control of parameter memory and computation during training and inference. While SS1 layers can be used to build a model itself, we can also obtain SS1 models from pre-trained models. This is especially important since many useful models are pre-trained once, and weights are open-sourced for public usage. For this purpose, we provide a projection function to project full matrices onto SS1 matrices, which can transfer knowledge from pre-trained models. Moreover, we demonstrate that SS1 can be effectively combined with post-training quantization to harness the advantages of both approaches. Additionally, we present theoretical insights explaining why the integration of SS1 and quantization leads to performance improvements that neither method can achieve independently.

We evaluate SS1 layers on a broad set of settings and use cases. A summary of our findings is below.

- SS1 has better quality-efficiency tradeoff than competing methods like Monarch[16] (state-of-the-art structured sparsity) and LowRank[2] across various domains. With SS1, we can build better models at lower parameter counts while delivering superior inference latency. For example, we achieve up to a $1.30\times$ improvement in GPT2 [17] model inference throughput.
- Pretrained models can be projected onto SS1 and further finetuned to deploy fast models. We show that we can maintain the GLUE[18] score of BERT[19] while speeding up inference by $1.31\times$.
- SS1 projected models can be used even without finetuning with reasonable accuracies. We show proof-of-concept benefits in the Llama-3-8B model[1] with $1.11\times$ faster inference.
- Quantization[2] is a highly effective technique for improving efficiency that can be combined with SS1. Not only do we see this empirically, but we can also see it in theory.
- SS1 also impacts CPU ML workloads significantly. For instance, we reduce the MLP workload of DLRM[20] MLPerf Model, which contributes over $70\%$ inference latency, by approximately $2\times$ using SS1 layers without compromising model quality.

## 2  Related Work and Background

**Unstructured Sparsity:** The redundancy in the deep learning model can be removed by sparsifying the model using iterative procedure [2, 3, 4]. Another related line of work is that of Lottery ticket hypothesis [5, 6, 21], which tries to find a sparse model at the start of training. Apart from expensive procedures to find these subnetworks, the unstructured sparsity still needs to deliver on the promise of latency improvements for inference.

**Structured Sparsity:** Linear transformations that are efficient and expressive has been an active line of research for over a decade. The general direction here is to create a combination of sparse, diagonal, permutation, and sub-linear transformations such as FFT, DCT, and Hadamard [14, 22, 13, 23, 24]. Some of them also used fixed random sparse/diagonal matrices. Structured matrices proposed in this line of research for transformation of size $K \times K$ have $O(K)$ parameters and $O(Klog(K))$ FLOPs. The actual speed-up obtained on modern hardware using these methods is limited. More importantly, the expressivity of these matrices is severely restricted due to very few learnable parameters. A

recent line of work exploits butterfly matrices [25, 16] and their variations, such as Monarch to obtain expressive transformations. These matrices also fit the general recipe of structured matrices specified above. However, a critical distinction in Monarch matrices is the presence of many more parameters. Specifically, the pure monarch matrices are supposed to contain $O(K\sqrt{K})$ parameters and FLOPs when using $\sqrt{K}$ blocks or factors (more blocks imply fewer parameters). However, practically, the matrices that have enough representative power in deep learning context generally use two or four blocks [16], i.e. $O(K^2)$ parameters and FLOPs. Nevertheless, Monarch shows great latency benefits for training with scheduled different-sized monarch decomposition. However, to our knowledge, the inference benefits for these matrices are limited. Our paper compares SS1 against Monarch since it is the SOTA representative for this class of matrices.

**Quantization:** Post training quantization [2] is currently one of the most successful tools for improving large models' efficiency. The basic idea is to reduce the precision of the weights and activations to reduce the memory footprint of the model and also exploit integer arithmetic [7] for better compute throughput. The literature on quantization is vast, and more details can be found in the survey [26]. In this paper, we show that we can combine SS1 with quantization techniques to further improve latency. We also explore the theory of combining quantization with SS1 approximation in section 4, which can be of independent interest.

**Randomized Parameter Sharing:** Randomized parameter sharing was first introduced in [8] as a general model compression tool. RPS is primarily used to reduce the parameter memory footprint of models. The parameter memory is separated from the model's actual computational graph. Each model's weight is mapped to the parameter memory using a random hash function. The value of the weight is then the value from the parameter accessed via hash functions. Essentially, if two weights are mapped to the same value in parameter memory, they are tied together and share a single learnable parameter.

The randomness of weight tying that leads to theoretical guarantees concerning projection quality was also the reason behind the extremely slow systemic performance of the proposal. The systemic performance was fixed using block-based hash mappings showing practical applications in various domains [10, 9, 27]. Surprisingly, the projection quality improves with block-based projections[10] leading to a strictly superior RPS system. RPS quality was further enhanced using global parameter sharing (parameter sharing across modules) which is superior to module-specific parameter sharing [9]. This, however, leads to additional challenges regarding the stability of training the model. These challenges were resolved in [15]. This work also proved a missing link in the theoretical analysis of RPS methods – it showed that the quality of dimensionality reduction (alt. projection quality) directly correlates with the quality of models learned under projection for linear models. Further, it was shown that random dense projections (which underlie RPS) are superior to random sparse projections (which underlie Pruning) which justifies why RPS methods convincingly outperform pruning methods, especially at high compressions in [15]. Interestingly, most of the RPS literature is strictly focused on reducing the memory footprint and does not affect the computational workload. In this work, we further improve the utility of RPS techniques by deploying them to reduce computation.

### 2.1 Background on Randomized Parameter Sharing required for SS1

**Tying of parameters using hash functions** $(h, g)$**:** Under RPS, there is a single parameter memory $\mathcal{M}, |\mathcal{M}| = m$, and each weight inside the model is mapped to one of the parameters in the memory using hash functions. Let the flattened weight vector of the entire model be $\theta, |\theta| = n$. A weight, say $\theta[i]$, is uniquely identified using a set of integers (module number, location inside the module, etc.), say $id(i) \in \mathbf{N}^k$ for some $k$, and then the value of the weight is

$$\theta[i] = g(id(i))\mathcal{M}[h(id(i))] \qquad h : \mathbf{N}^k \to [m], g : \mathbf{N}^k \to \{\pm 1\}$$

where $[m] = \{0, 1, ..., m-1\}$.

**Sketch representation of recovery:** We can write the entire recovery of the vector $\theta$ as a linear projection from $\mathcal{M}$ using a sketch matrix $S \in R^{n \times m}$ which is defined as

$$\forall i \in [n], S[i, h(id(i))] = g(id(i)), \qquad \forall j \neq h(id(i)), S[i, j] = 0 \qquad (1)$$

Then the weight $\theta$ and the parameter memory $\mathcal{M}$ is related by, $\theta = S\mathcal{M}$

**Dimensionality reduction problem:** Traditionally, random projections are used as a dimensionality reduction technique for data points. The projection is considered better if it can maintain the structure of the dataset (inter-point distances or equivalently inner products between points). The

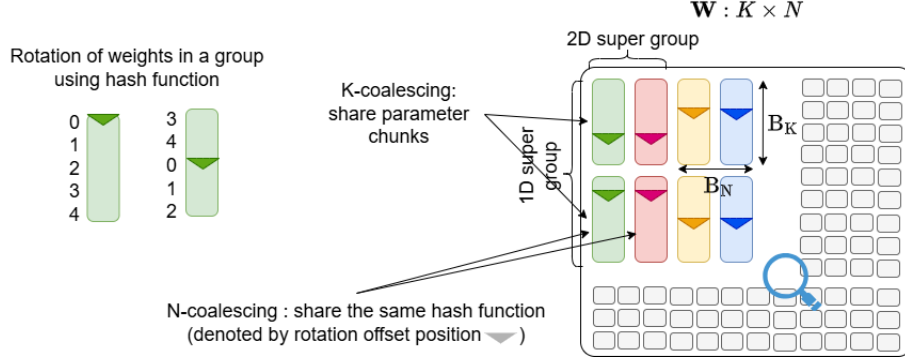

Figure 1: Illustration of weight tying in SS1. Same colored weights imply that they are tied to the same parameter in memory

standard theoretical setup is as follows. Consider two arbitrary vectors $x, y \in R^n$, Consider a projection function PROJ (or compression technique in general), then $\hat{x}, \hat{y}$ are projected vectors, $\hat{x} = \text{PROJ}(x), \hat{y} = \text{PROJ}(y)$. Then, the quality of the projection is measured by

$$\text{MSE}(\langle \hat{x}, \hat{y} \rangle) = \mathbf{E}\left(|| \langle x, y \rangle - \langle \hat{x}, \hat{y} \rangle ||_2^2\right) \tag{2}$$

where the expectation is over the randomization scheme of the projection. This problem is of interest to evaluate RPS-based learning since the quality of the learned model under projection $f(S\mathcal{M})$ correlates with the quality of data projection $S^\top$ for the standard dimensionality reduction problem with $\hat{x} = S^\top x$ and $\hat{y} = S^\top y$ [15].

## 3   Sketch Structured Transform(SS1)

The SS1 layer falls into the category of RPS methods. We will use small case letters for scalars, boldface small cases for vectors, and boldface capitals for matrices. We use numpy notation of indexing. For instance $\mathbf{z}[(\text{range}(n) + 5)\%n][i : j]$ first rotates the vector $\mathbf{z}$ by 5 places and then selects a subarray from $i$ to $j$.

### 3.1   Parameter sharing in SS1

In standard RPS, where the weights are mapped into a single memory $\mathcal{M}$, with a high probability, weights corresponding to a single neuron are not tied together. While this seems advantageous for expressivity, it is also the reason why RPS methods cannot reduce computation since we need to perform $O(K)$ multiplications for a single neuron. In SS1, we perform restricted parameter sharing where parameters are tied inside a single neuron only. Consider a linear transform $\mathbf{y}^\top = \mathbf{x}^\top \mathbf{W}$ where $\mathbf{x} \in \mathbb{R}^K$, $\mathbf{y} \in \mathbb{R}^N$ and $\mathbf{W} : \mathbb{R}^{K \times N}$ matrix.

**Single Neuron RPS:** Let us consider a single neuron $y = \mathbf{x}^\top \mathbf{w}$. Under SS1, the weights $\mathbf{w} \in \mathbb{R}^K$ come from compressed parameter vector $\mathbf{z} \in \mathbb{R}^{K//c}$ where $c$ is the compression factor. We use $//$ to denote integer division. For simplicity, we assume throughout the section that $c$ is an integer and $c|K$. Each neuron has its own $\mathbf{z}$. Weights $\mathbf{w}$ are recovered from $\mathbf{z}$ using standard RPS. i.e.,

$$\mathbf{w}[i] = g(i)\mathbf{z}[h(i)] \tag{3}$$

Equivalently, $\mathbf{w}$ can be represented as

$$\mathbf{w} = \mathbf{S}\mathbf{z} \tag{4}$$

where $\mathbf{S} : K \times K//c$ is sparse matrix according to Equation 1. When parameters are shared in this manner, the computation can be reduced.

$$y = \mathbf{x}^\top \mathbf{w} = \mathbf{x}^\top (\mathbf{S}\mathbf{z}) = \left(\mathbf{x}^\top \mathbf{S}\right) \mathbf{z} \tag{5}$$

Thus computing $y$ needs only $K//c$ multiplications since $\mathbf{x}^\top \mathbf{S}$, a sketch of input $\mathbf{x}$, can be implemented using only additions and subtractions. As is, this randomized mapping will not be cache-efficient and thus is not GPU-friendly. We show how to make it GPU-friendly using $K$ and $N-$coalescing in the subsequent parts of this section.

$K-$**coalescing:**   The single-neuron computation shown above is not GPU-friendly for arbitrary choice of RPS mapping hash function $h$. We now explain the hash function $h$ used in SS1.

The parameter tying is illustrated in Figure 1. We first divide the weight vector $\mathbf{w}$ into chunks or groups of size $B_K$, a hyper-parameter. The chunk-id and offset inside the chunk of weight can be written as,

$$\mathcal{C}(i) = i//B_K \qquad \mathcal{O}(i) = i\%B_K \tag{6}$$

Then $c$ ( recall that $c$ is the integral compression factor ) chunks are grouped together into 1D supergroups. The 1D super group ID of the weight can be written as,

$$\mathcal{G}_1(i) = \mathcal{C}(i)//c \tag{7}$$

We want to restrict the hash function in SS1 to ensure the following,

1. Each chunk of size $B_K$ is contiguously located in $\mathbf{z}$ and do not share weights.
2. chunks that belong to the same 1D supergroup share weights among each other.

We use the following mapping $h' : [K] \rightarrow [K//c]$ to satisfy both conditions. Let $h : \mathbf{N} \times \mathbf{N} \rightarrow \mathbf{N}$ be a universal hash function. Then,

$$h'(i) = \mathcal{G}_1(i)B_K + (h(\mathcal{G}_1(i), \mathcal{C}(i)) + \mathcal{O}(i))\%B_K \tag{8}$$

Note that many hash functions which follow the two conditions exist. However, we will stick with this hash function since this is the function we implement. With this hash function, the $j^{th}$ chunk of $i^{th}$ 1D super group in $\mathbf{w}$ can be written as,

$$\mathbf{w}[(ic+j)B_K : (ic+j+1)B_K] = \mathbf{z}[iB_K : (i+1)B_K][(\mathcal{R}(0 : B_K) + h(i,j))\%B_K] \tag{9}$$

where $\mathcal{R}(0 : n) = (0, 1, ..., n - 1)$ is the range function. In matrix form using Equation 1, let $\mathbf{S}_{i,j} : B_K \times B_K$ be the matrix representation of above hash function restricted to $j^{th}$ chunk of $i^{th}$ 1D supergroup. Then,

$$\mathbf{w}[(ic+j)B_K : (ic+j+1)B_K] = \mathbf{S}_{i,j}\mathbf{z}[iB_K : (i+1)B_K] \tag{10}$$

Then, we can write the single neuron computation as,

$$y = \mathbf{x}^\top\mathbf{w} = \sum_{i=0}^{(K//B_K//c)-1} \left( \sum_{j=0}^{c-1} \mathbf{x}^\top[(ic+j)B_K : (ic+j+1)B_K]\mathbf{S}_{i,j} \right) \mathbf{z}[iB_K : (i+1)B_K] \tag{11}$$

We can implement the above computation as follows: Bring in one chunk of $\mathbf{z}$; then bring in $c$ chunks belonging to the 1D super group of $\mathbf{x}$ one at a time and aggregate them into a single chunk of size $B_K$ using sketch matrices $\mathbf{S}$; perform the dot product on the chunk of $\mathbf{z}$ and aggregated chunk of $\mathbf{x}$ and move on to the next chunk of $\mathbf{z}$ and 1D supergroup of $\mathbf{x}$. Note that we can perform this operation in a block manner and using exactly one read of $\mathbf{z}$ and $\mathbf{x}$ due to the nature of the hash function.

$N-$**coalescing:** Now let us move our discussion from single neuron computation in SS1 to computing all the neurons, i.e. the entire matrix multiplication. In this case we have the complete weight matrix $\mathbf{W} : K \times N$ which is derived from a compressed matrix $\mathbf{Z} : K//c \times N$. Apart from $K$- coalescing specific to each neuron, to fully utilize the GPU capabilities, we further restrict parameter sharing along the $N$ dimension. The weight mapping is illustrated in Figure 1. We divide the weights along the $N$ dimension in column blocks of size $B_N$. Each neuron that belongs to the same block will have the same set of hash functions. This allows us to do a block-based computation as,

$$\mathbf{y}[: B_N] = \mathbf{x}^\top\mathbf{W}[:, : B_N] =$$
$$\sum_{i=0}^{(K//B_K//c)-1} \left( \sum_{j=0}^{c-1} \mathbf{x}^\top[(ic+j)B_K : (ic+j+1)B_K]\mathbf{S}_{i,j} \right) \mathbf{Z}[iB_K : (i+1)B_K, : B_N] \tag{12}$$

Note that each neuron has its own parameter space in $\mathbf{Z}$. The complete algorithm is presented in Algorithm 1. It considers batched multiplication and uses $B_M$ hyper-parameter, which is standard in matrix multiplication. Also, it uses a slightly different notation for ease of expression. However, it maintains the principles introduced in this section. The kernel implementation and optimization details are deferred to Appendix E for lack of space.

## 3.2  SS1 projection from the pre-trained model

A pre-trained model can be projected into SS1 and fine-tuned for downstream tasks. The complete model projection boils down to projecting each full matrix in a linear layer, say $\mathbf{W} : K \times N$ into parameter tied SS1 matrix $\mathbf{Z} : (K//c) \times N$. Recall that SS1 has parameter sharing independent in each neuron. Thus, we just have to find the projection for each neuron's weight. Consider the neuron

**Algorithm 1** SS1(Z, X)

---

**Require:** $\mathbf{X} \in \mathbb{R}^{M \times K}$: data matrix, $c \in \mathbb{N}$: integral compression factor, $\mathbf{Z} \in \mathbb{R}^{(K//c) \times N}$: weight matrix, $B_M, B_K, B_N$: coalescing parameters, $h : \mathbb{N}^3 \to \{0, \ldots, B_K - 1\}, g : \mathbb{N}^3 \to \{\pm 1\}$
**Require:** $cB_K | K$
**Ensure:** $\mathbf{Y} \in \mathbb{R}^{M \times N} = \text{SS1}(\mathbf{X}, \mathbf{Z})$

1: $\mathbf{T}_x = \text{TILE}(B_M, B_K)$, $\mathbf{T}_y = \text{TILE}(B_M, B_N)$ ▷ Allocate 2D tiles. $\mathbf{T}_x$ will store the intermediate input sketch and $\mathbf{T}_y$ will store intermediate result
2: **for** $i \in [\lceil M/\mathbf{B_M} \rceil]$ **do**
3:      **for** $j \in [\lceil N/\mathbf{B_N} \rceil]\}$ **do**
4:          $\mathbf{T}_y[:, :] = 0$                                                    ▷ reset output tile
5:          **for** $k \in [\lceil K/c \rceil]\}$ **do**
6:              $\mathbf{T}_x[:, :] = 0$                                      ▷ reset input tile
7:              **for** $l \in [c]$ **do**                      ▷ Iterate over $c$ blocks to create input sketch
8:                  $a = \mathbf{X}[iB_M : (i+1)B_M, (kc+l)B_K:(kc+l+1)B_K]$ ▷ Bring in one chunk of input
9:                  $a = g(j, k, l)a[:, (\mathcal{R}(0 : B_K) + h(j, k, l))\% B_K]$ ▷ Rotate input by a random offset
10:                $\mathbf{T}_x + = a$                                   ▷ Aggregate the sketch
11:              **end for**
12:              $\mathbf{T}_y = \mathbf{T}_y + \mathbf{MM}(x, Z[kB_K:(k+1)B_K, jB_N:(j+1)B_N])$ ▷ Fetch block of $\mathbf{Z}$, perform MM with sketched input and aggregate in tile $\mathbf{T}_y$
13:          **end for**
14:          $\mathbf{Y}[iB_M:(i+1)B_M, jB_N:(j+1)B_N] = \mathbf{T}_y$
15:      **end for**
16: **end for**

---

$y = \mathbf{x}^\top \mathbf{w}$ and corresponding compressed with $\mathbf{z}$ and recovery sketch matrix $\mathbf{S}$. Given $\mathbf{w}$, we want to find $\mathbf{z}$ that minimizes

$$\mathbf{z}^* = \text{argmin}_z ||\mathbf{w} - \mathbf{S}\mathbf{z}||_2 \tag{13}$$

The solution is that of linear regression, $\mathbf{z}^* = (\mathbf{S}^\top \mathbf{S})^{-1} \mathbf{S}^\top \mathbf{w}$. Note that $(\mathbf{S}^\top \mathbf{S})$ is a diagonal matrix with non-zero diagonal elements. Once we solve for the value of $\mathbf{z}$ for each neuron in each weight matrix, we have our overall projection onto SS1. The algorithm for projection can also be implemented in a blocked manner and is given in Algo 2 in Appendix G

### 3.3 SS1 combined with standard RPS.

SS1, which reduces the computation of the linear layers, can be combined with RPS methods, which reduce the parameter memory to create a holistic efficiency system – one that lets you control both memory and computation independently. The only change to Algo 1 is in where we read the weight tile. In the current algorithm, we maintain $\mathbf{Z} : K//c \times N$, the compressed weight matrix, in a row-major format. Under RPS, we can locate the tile $\mathbf{Z}[kB_K : (k+1)B_K, jB_N : (j+1)B_N]$ in the RPS memory, say $\mathcal{M}$, using another hash function, say $h'$, at $\mathcal{M}[h'(k, j)\% |\mathcal{M}| - B_K B_N]$.

## 4 Theoretical aspects of SS1

**Quantization + SS1 can beat the individual methods:** In this section, we analyze a combination of quantization and SS1 (i.e., projection) in the standard dimensionality reduction setup. We consider stochastic integer quantization for our analysis. Consider two vectors $\mathbf{x}, \mathbf{y} \in \mathbb{R}^n$. We aim to reduce the memory by a factor of $c$. In case of quantization, assuming initial precision as $F$, $c = b/F$ if we are using b-bit quantization. Similarly, for projection, $c = m/n$ if projecting the data to $\mathbb{R}^m$. In case of projection the compressed vector can be represented:

$$\hat{\mathbf{x}}_p[i] = \sum_{j=1}^{n} (g(j)\mathbf{1}(h(j)=i)\mathbf{x}[i]) \tag{14}$$

Let the component values be restricted to $(-D/2, D/2)$ for some $D$. The compressed vector in case of stochastic quanti-

Figure 2: Upper bound on variance

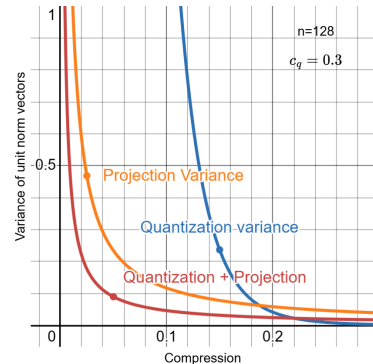

zation can be written as,

$$\hat{\mathbf{x}}_q[i] = \lfloor \mathbf{x}[i] \rfloor_q + \mathbf{1} \quad \text{where } \mathbf{1} = \begin{cases} 1 \text{ with prob } p = \frac{(\mathbf{x}[i] - \lfloor \mathbf{x}[i] \rfloor_q)}{D/2^b} \\ 0 \text{ with prob } (1-p) \end{cases} \tag{15}$$

The range $D$ is divided into $2^b$ buckets and $\lfloor . \rfloor_q$ rounds the value down in the buckets. The variance of the inner product between $\langle \hat{\mathbf{x}}, \hat{\mathbf{y}} \rangle$ for projection, quantization, and combination that first performs b-bit quantization followed by projection is given below,

**Theorem 1** *Consider two arbitrary vectors $\mathbf{x}, \mathbf{y} \in R^n$ such that $||\mathbf{x}|| \leq 1, ||\mathbf{y}|| \leq 1$ in F precision. For some fixed value $k$ in $[1, 2]$,*

$$Q(c) = \frac{2}{2^{2cF}} + \frac{n}{2^{4cF}} \quad P(c) = \frac{k}{cn} \tag{16}$$

*The variance of quantization ($V(\langle \hat{\mathbf{x}}, \hat{\mathbf{y}} \rangle_q)$), projection ($V(\langle \hat{\mathbf{x}}, \hat{\mathbf{y}} \rangle_p)$) with compression c, can be tightly bounded by*

$$V(\langle \hat{\mathbf{x}}, \hat{\mathbf{y}} \rangle_q) \leq Q(c) \quad V(\langle \hat{\mathbf{x}}, \hat{\mathbf{y}} \rangle_p) \leq P(c) \tag{17}$$

*The variance of combination with compression $c = c_p c_q$ with quantization with compression $c_q$ followed by projection with $c_p$ is bounded by*

$$V(\langle \hat{\mathbf{x}}, \hat{\mathbf{y}} \rangle_{pq}) \leq Q(c_q) \left( 1 + \frac{1}{c_p} \right) + P(c_p) \tag{18}$$

The best way to understand this theorem is to plot the variance as a function of compression as shown in the Figure 2. In the plot, we use $c_q = 0.3, n = 128$ and $k = 1.5$ for illustration. As we can see with the combination, the variance is better than individual methods in certain regions. The reason is that both methods do quite well in the low compression regime, and explode steeply at very high compression. Thus, when we combine the two methods, we can remain in relatively low error zones for both methods while obtaining greater overall compression.

**Analyzing parameters of SS1:** This section discusses the new aspects of SS1 and how it affects the quality of SS1. Specifically, we discuss the effect of parameters $B_K$ and $B_N$. From a latency viewpoint, it is natural that these parameters should be autotuned to optimize for latency, which is standard for matrix multiplication. However, since these parameters affect the weight tying, it is important to investigate their effect on the quality of models. We consider the standard dimensionality reduction setup to evaluate the changes in mapping performed by SS1. We find that $B_K$ is a pure latency parameter with no impact on learning quality. We present this as the following result.

**Theorem 2** *Consider two vectors $\mathbf{x}, \mathbf{y} \in R^n$ in higher dimensional space. Also, let $h$ and $g$ be the SS1 hash functions used to project these vectors to $\hat{\mathbf{x}}, \hat{\mathbf{y}} \in R^m$ ($m < n$). Let $B_K$ be the K-coalescing factor of $h$. Then, under compression $c = n/m$ and assuming $cB_K|m|n$, the inner product estimation is unbiased and has variance,*

$$V(\langle \hat{\mathbf{x}}, \hat{\mathbf{y}} \rangle) = \frac{1}{B_K} \sum_{i,j=1}^{n} \sum_{j=1, j \neq i}^{n} I(i,j) \left( x_i^2 y_j^2 + x_i y_i x_j y_j \right) \tag{19}$$

*where $I(i, j)$ is a Kronecker delta function indicating if $i$ and $j$ belong to the different groups in the super group of size $cB_K$. Under permutation $p$ of elements of $x$ and $y$ before projection,*

$$\mathbf{E}_p \left( V(\langle \hat{\mathbf{x}}, \hat{\mathbf{y}} \rangle - \langle \mathbf{x}, \mathbf{y} \rangle) \right) = \frac{(c-1)}{(n-1)} \sum_{i=1}^{n} \sum_{j=1, j \neq i}^{n} \left( x_i^2 y_j^2 + x_i y_i x_j y_j \right) \tag{20}$$

In the above theorem, we find that the quality of projection is unaffected by the parameter $B_K$. This can be understood by considering a specific element in $\mathbf{x}$. The number of other elements with which it can interact under projection depends only on the compression factor $c = n/m$ and not on $B_K$. We know from [15], that the quality of projection is directly related to the quality of learned linear models under compression. Thus, $B_K$ does not have any impact on learning quality as well. We analyse the effect of $B_N$ as well. But the analysis is deferred to the appendix (see section F.2) for lack of space.

Table 1: This table presents the quality and latency of NLP and vision experiments. Overall across both the domains, SS1 gives best quality models under similar parameters and better quality per unit compute. In terms of inference latency, we see upto $1.3\times$ increase in throughput. Exact experimental details are present in Appendix I.

| Model | #params | PPL (GPT2-S) | loss (GPT2-S) | Model Latency (GPT2-S) | | Model Latency (GPT2-M) | | Model Latency (GPT2-L) | |
|---|---|---|---|---|---|---|---|---|---|
| | | | | b=16 | b=32 | b=16 | b=32 | b=8 | b=16 |
| Original Model | 124M | 20.43 | 3.004 | 80.4 | 160 | 181 | oom | 190 | 376 |
| SS1-2x | 96M | **19.458** | **2.957** | 78.5 | 154.9 | 172.4 | 342.7 | 169.9 | 334.2 |
| SS1-4x | 81M | **19.998** | **2.991** | 75.1 | 148.6 | 163.3 | 330 | 157.6 | 310.4 |
| SS1-8x | 74M | **20.68** | **3.025** | 74.3 | 145.6 | 153 | 309.3 | 147.9 | 291 |
| Monarch (nb=2) | 103M | 19.606 | 2.964 | 90 | 178.2 | 201.8 | 401.6 | 194.8 | 386.5 |
| Monarch (nb=4) | 85M | 20.611 | 3.022 | 88.2 | 174.2 | 186.9 | 371 | 178 | 363.7 |
| Monarch (nb=8) | 76M | 22.839 | 3.119 | 85.5 | 169.4 | 181.1 | 359.4 | 173.6 | 342.8 |
| Lowrank~2x | 96 | 19.756 | 2.979 | 99 | 195.2 | 231.5 | 456.3 | 150 | 295.3 |
| Lowrank~4x | 81M | 20.152 | 2.993 | 84.8 | 165.6 | 159.1 | 314.2 | 137 | 266.6 |
| Lowrank~8x | 74M | 21.211 | 3.05 | 72.2 | 141.4 | 147.3 | 290.7 | 131 | 259 |

(a) GPT models on wikitext-103 (standard deviation : around 0.2PPL measured for baseline model)

| Model | #params | Accuracy (C100) | Accuracy (C10) | Model Latency (MM-S) | | Model Latency (MM-M) | | Model Latency (MM-L) | |
|---|---|---|---|---|---|---|---|---|---|
| | | | | b=512 | b=1K | b=512 | b=1K | b=512 | b=1K |
| Original Model | 1.1M | 0.6751 | 0.9083 | 7.9 | 15.4 | 38.1 | 75.4 | 122.5 | 246.1 |
| SS1-2x | 613K | **0.6692** | **0.8918** | 8.3 | 14.7 | 36.2 | 71.7 | 114.3 | 227.3 |
| SS1-4x | 347K | **0.6463** | **0.8695** | 8.2 | 14.7 | 33.6 | 66.7 | 100.4 | 199.2 |
| SS1-8x | 282K | **0.6068** | **0.8554** | 8.2 | 14.5 | 33.2 | 65.5 | 95.0 | 188.1 |
| Monarch (nb=2) | 750K | **0.6738** | 0.8729 | 12.5 | 20.8 | 47.7 | 94.4 | 135.9 | 272.9 |
| Monarch (nb=4) | 422K | 0.6261 | 0.8491 | 12.9 | 21.3 | 46.2 | 91.7 | 119.9 | 239.9 |
| Monarch (nb=8) | 259K | 0.5731 | 0.8282 | 14.2 | 23.4 | 46.0 | 91.4 | 115.3 | 229.8 |
| Lowrank~2x | 750K | 0.6628 | 0.8907 | 9.7 | 18.4 | 38.6 | 78.1 | 114.6 | 231.7 |
| Lowrank~4x | 422K | 0.6299 | 0.8639 | 9.5 | 18.1 | 35.9 | 72.0 | 100.6 | 204.2 |
| Lowrank~8x | 259K | 0.5353 | 0.8034 | 9.4 | 17.9 | 35.6 | 71.3 | 94.9 | 191.1 |

(b) MLPMixer models on CIFAR datasets (Stdev: 0.001 (2x), 0.002 (4x), 0.02 (8x))

Table 2: PPL(loss) for **[Left]:** Applying quantization on some saved checkpoints of original and SS1 models. The effect of quantization on SS1 is similar to that on full model. **[Right]**: SS1 models can outperform standard models. The std-deviation is around 0.2PPL for these experiments.

| | Quantization + SS1 | | | | SS1 vs. Small | | |
|---|---|---|---|---|---|---|---|
| Model | #param | Before Quantization | After Quantization | | Model | #param | Quality |
| GPT-S | 124M | 21.228 (3.013) | 21.25 (3.014) | | GPT2-S (Small-4x) | 81M | 19.71 (2.977) |
| GPT-S (SS1-4x) | 81M | 20.497 (2.986) | 20.53 (2.988) | | GPT2-S (Small-6x) | 76M | 22.14 (3.087) |
| GPT-S (SS1-8x) | 74M | 21.228 (3.023) | 21.26 (3.025) | | GPT-S (Small-4x-SS1-2x) | 74M | 20.50 (3.01) |

# 5 Experiments

## 5.1 Accuracy vs. Latency evaluation of SS1

We first evaluate the expressiveness and latency of our SS1 against popular baselines of Monarch, SOTA structured sparsity-based transformation, and LowRank transformation. We benchmark various shapes for the "nn.Linear", Monarch, LowRank, and SS1 kernels to evaluate the standalone kernel latency. We use the official Monarch implementation by the authors[4] with number of blocks (nb) as the compression controlling factor. The results are deferred to the appendix due to space constraints. We also measure end-to-end model latency improvement for various sizes of GPT2 and MLPMixer (MM) [28] models. We show the latency of three sizes of models (small: S, medium:M,

Table 3: **[Left:]**BERT pretrained model projected onto SS1 and finetuned on GLUE benchmark. Due to the descrepancy, we report both online and our local results of full model. The SS1 model is $1.31\times$ **faster** for higher batch sizes. Details are available in I.2 **[Right:]** Proof of concept of applications in Llama-3-8B. We obtain $1.11\times$ speed up in latency by compressing selective layers, without any form of retraining or finetuning. Details of the experiment are in I.3

| | **SS1 projection followed by finetuning** | | | | | **SS1 projection without finetuning** | | |
|---|---|---|---|---|---|---|---|---|
| | BERT-LARGE | | BERT-BASE | | | LLAMA-3-8B | | |
| | #par | GLUE score | #par | GLUE score | | #par | MMLU | WINO |
| baseline (reported) | 335M | 80.4 | 109M | 78.6 | | | | |
| baseline (our run) | 335M | 82.2 | 109M | 82.16 | baseline (our run) | 8B | 65.05 ($\pm$ 0.0038) | 76.1 ($\pm$ 0.011) |
| SS1 | 181M | 79.6 ($\pm$ 0.203) | 66M | 79.9 ($\pm$ 0.066) | SS1 | 6B | 61.26 ($\pm$ 0.0039) | 69.93 ($\pm$ 0.012) |

Table 4: The time spent by 100GB sized DLRM MLperf benchmark model in various components. Specifically 70% latency is spent in Top MLP. We can reduce MLP workload by factor of $2\times$ by training SS1 layers without compromising quality of the model. (Statistical significance: multiple runs gives 0.8032)

| | Bottom MLP | Interaction | Embedding Lookup | Top MLP | Top MLP params | Quality |
|---|---|---|---|---|---|---|
| DLRM-MLPerf | 6.63% | 12.3% | 9.7% | 71.3% | 1.7M | 0.8032 |
| DLRM-MLPerf $\times$ SS1-2x | — needs CPU kernel implementation — | | | | 0.9M | 0.8032 |

and large:L)for both architectures. Exact details are provided in appendix I) Additionally, we perform quality experiments in end-to-end training using these matrices for two domains: (a) NLP: GPT2-S model on wikitext-103 dataset [29] using test perplexity (PPL) and loss as the metric (b) Vision: MLPMixer on CIFAR (C10, C100)[30] and Tiny-Imagenet datasets [31] using accuracy as the metric. The results are presented in Table 1. The details of these experiments can be found in the Appendix I. We make the following observations.

- In our kernel latency evaluation, SS1 consistently outperforms Monarch and LowRank under similar parameter budgets (Table 4 in appendix). Monarch is generally worse than full multiplication except in larger shapes. LowRank is competitive with SS1 for higher compression.
- The kernel-level latency benefits also translate to end-to-end model latency (Table 1 and Table 1b). This can be seen across both Transformer and MLPMixer architectures, two of the SOTA architectures for deep learning. Although, at times, LowRankLinear gives better performance at higher compression, the quality of the model obtained is inferior to SS1. With larger models, we see an increase of around $1.3\times$ in end-to-end throughput with SS1. It also enables us to run larger batches.
- In terms of quality, SS1 consistently outperforms Monarch and LowRank across NLP and Vision domains on GPT and MLPMixer architectures under similar parameter budgets.
- Overall, we see that SS1 allows better quality per unit compute than its structured competitors.

***Building expressive and faster models:*** Should we build and train models with SS1 layers instead of standard linear layers? Although answering this question requires a broader evaluation, we cautiously believe that the answer might be yes. We can indeed obtain better quality models by using SS1 layers instead of "nn.Linear" layer. This is demonstrated in Table 2. The quality of the SS1 model with 74M parameters is better than that of standard models with similar or more parameters.

## 5.2   SS1 + Quantization

Quantization is an important efficiency technique to improve inference time memory and latency. Although SS1 already reduces the memory footprint and improves inference latency, it can be combined with quantization to obtain further inference benefits. One might wonder if combining two approximations (SS1 and quantization) can lead to worse-quality models. We show that SS1 can be combined with quantization without significantly impacting quality (also supported by theory) In fact, the effect of quantization on SS1 is similar to the full standard model. The quality details before

Table 5: PPL(latency) for GPT-2(124M) model trained with different compressions SS1 and ROAST

| | PPL(Latency) | | |
|---|---|---|---|
| | 2x | 4x | 8x |
| SS1 | 19.45(154) | 19.99(148) | 20.68(145) |
| ROAST | 19.87(238) | 20.20(237) | 20.94(222) |

and after quantization are in Table 2. To measure the latency benefits, we would have to implement quantized SS1 kernels, which is out of the scope of this paper.

### 5.3 Dense-to-structured finetuning of pre-trained models for efficiency

We show that pre-trained models can be projected onto SS1-structured space and finetuned for downstream tasks. We demonstrate this by projecting selected linear transformations in BERT to SS1 and training the resulting model on the GLUE Benchmark. We obtain a $1.31\times$ faster model which gives a similar GLUE score in downstream tasks. More details are mentioned in I. The results are summarized in Table 3

We find that we can improve inference of Llama-3-8B model for downstream tasks if we project carefully selected layers onto SS1. The selection is done on a calibration dataset that corresponds to the task at hand. We show that, even without finetuning, we can reduce the computation involved in the model while maintaining reasonable accuracy. Results can potentially improve if we finetune post-projection as we see in BERT. The results are summarized in Table 3.

### 5.4 Improving CPU workloads e.g. DLRM MLPerf Benchmark

While this paper focuses on providing GPU kernel for SS1, the algorithm can also be utilized to improve the inference performance of CPU deployments. Specifically, recommendation workloads of models such as the Deep learning recommendation model (DLRM) are often run on CPUs due to humungous embedding tables. We consider the DLRM MLPerf Benchmark, which has 100GB embedding tables and 10MB MLP layers. Nevertheless, on CPUs, the latency bottleneck is that of matrix multiplications. Specifically, 71.3% of the time is spent in the top MLP component (see Table 4). We show that we can train from scratch a SS1 version of DLRM, which has half the parameters in MLP and maintains the quality of the full model. While CPU-kernel implementation of SS1 is out of the scope of this paper, we expect that benefits shown on GPUs will also translate to CPUs, and thus, with SS1, we can improve the throughput of DLRM MLPerf Benchmark on CPUs without compromising the quality of the model.

### 5.5 Comparison with standard RPS method

We also compare SS1 with standard RPS method. Since SS1 focuses on compressing linear layers, we compare it with ROAST-MM compression with each matrix having its individual separate memory. The results are presented in Table 5. As expected, RPS methods do not improve the latency.

**Limitations:** Our work is mainly limited on two aspects. On the theoretical side, our analysis of parameters and the combination of quantization with SS1 is conducted mainly over linear models, which is a reasonable starting point. However, these results may not necessarily extend to deep learning models, and we plan to explore this in future research. Additionally, regarding our SS1 proposal, the efficiency gains are marginal beyond an $8\times$ compression of parameters. In future work, we aim to develop efficient structures where the efficiency gains are more closely aligned with the degree of parameter reduction.

## 6 Conclusion

In this paper, we introduce an efficient structured linear transformation termed SS1. We demonstrate that SS1 outperforms state-of-the-art structured baselines in terms of quality per unit of computation. Additionally, we show that SS1 provides immediate benefits across diverse applications, including language understanding, generative AI, and recommendations. Furthermore, we illustrate how SS1 can be combined with the popular quantization approach to achieve further improvements. We also theoretically explain the underlying reasons that may be of independent interest.

## 7 Acknowledgments

This work was supported by National Science Foundation SHF-2211815 and grants from Adobe, Intel, Total, and VMware.

## Footnotes

*Equal contribution; order chosen by a flip of a coin

†Work done in sole affiliation with George Mason University

[3]https://github.com/apd10/Sketch-Structured-Linear/

[4]https://github.com/HazyResearch/fly/blob/master/src/models/layers/monarch_linear.py

[5]https://pypi.org/project/pytorch-block-sparse/

## References

[1] Hugo Touvron, Thibaut Lavril, Gautier Izacard, Xavier Martinet, Marie-Anne Lachaux, Timothée Lacroix, Baptiste Rozière, Naman Goyal, Eric Hambro, Faisal Azhar, et al. Llama: Open and efficient foundation language models. *arXiv preprint arXiv:2302.13971*, 2023.

[2] Song Han, Huizi Mao, and William J Dally. Deep compression: Compressing deep neural networks with pruning, trained quantization and huffman coding. *ICLR*, 2016.

[3] Song Han, Jeff Pool, John Tran, and William Dally. Learning both weights and connections for efficient neural network. *Advances in neural information processing systems*, 28, 2015.

[4] Song Han, Jeff Pool, Sharan Narang, Huizi Mao, Enhao Gong, Shijian Tang, Erich Elsen, Peter Vajda, Manohar Paluri, John Tran, et al. Dsd: Dense-sparse-dense training for deep neural networks. *arXiv preprint arXiv:1607.04381*, 2016.

[5] Jonathan Frankle and Michael Carbin. The lottery ticket hypothesis: Finding sparse, trainable neural networks. *arXiv preprint arXiv:1803.03635*, 2018.

[6] Jonathan Frankle, Gintare Karolina Dziugaite, Daniel M Roy, and Michael Carbin. Stabilizing the lottery ticket hypothesis. *arXiv preprint arXiv:1903.01611*, 2019.

[7] Benoit Jacob, Skirmantas Kligys, Bo Chen, Menglong Zhu, Matthew Tang, Andrew Howard, Hartwig Adam, and Dmitry Kalenichenko. Quantization and training of neural networks for efficient integer-arithmetic-only inference. In *Proceedings of the IEEE conference on computer vision and pattern recognition*, pages 2704–2713, 2018.

[8] Wenlin Chen, James Wilson, Stephen Tyree, Kilian Weinberger, and Yixin Chen. Compressing neural networks with the hashing trick. In *International conference on machine learning*, pages 2285–2294. PMLR, 2015.

[9] Aditya Desai, Keren Zhou, and Anshumali Shrivastava. Hardware-Aware Compression with Random Operation Access Specific Tile (ROAST) Hashing. In *International Conference on Machine Learning*, pages 7732–7749. PMLR, 2023.

[10] Aditya Desai, Li Chou, and Anshumali Shrivastava. Random Offset Block Embedding (ROBE) for compressed embedding tables in deep learning recommendation systems. In D. Marculescu, Y. Chi, and C. Wu, editors, *Proceedings of Machine Learning and Systems*, volume 4, pages 762–778, 2022.

[11] Tara N Sainath, Brian Kingsbury, Vikas Sindhwani, Ebru Arisoy, and Bhuvana Ramabhadran. Low-rank matrix factorization for deep neural network training with high-dimensional output targets. In *2013 IEEE international conference on acoustics, speech and signal processing*, pages 6655–6659. IEEE, 2013.

[12] Trevor Gale, Erich Elsen, and Sara Hooker. The state of sparsity in deep neural networks. *arXiv preprint arXiv:1902.09574*, 2019.

[13] Marcin Moczulski, Misha Denil, Jeremy Appleyard, and Nando de Freitas. Acdc: A structured efficient linear layer. *arXiv preprint arXiv:1511.05946*, 2015.

[14] Quoc Viet Le, Tamás Sarlós, and Alexander Johannes Smola. Fastfood: Approximate kernel expansions in loglinear time. *arXiv preprint arXiv:1408.3060*, 2014.

[15] Aditya Desai and Anshumali Shrivastava. In defense of parameter sharing for model compression. In *The Twelfth International Conference on Learning Representations*, 2024.

[16] Tri Dao, Beidi Chen, Nimit S Sohoni, Arjun Desai, Michael Poli, Jessica Grogan, Alexander Liu, Aniruddh Rao, Atri Rudra, and Christopher Ré. Monarch: Expressive structured matrices for efficient and accurate training. In *International Conference on Machine Learning*, pages 4690–4721. PMLR, 2022.

[17] Alec Radford, Jeff Wu, Rewon Child, David Luan, Dario Amodei, and Ilya Sutskever. Language models are unsupervised multitask learners. 2019.

[18] Alex Wang, Amanpreet Singh, Julian Michael, Felix Hill, Omer Levy, and Samuel R Bowman. Glue: A multi-task benchmark and analysis platform for natural language understanding. *arXiv preprint arXiv:1804.07461*, 2018.

[19] Jacob Devlin, Ming-Wei Chang, Kenton Lee, and Kristina Toutanova. Bert: Pre-training of deep bidirectional transformers for language understanding. *arXiv preprint arXiv:1810.04805*, 2018.

[20] Maxim Naumov, Dheevatsa Mudigere, Hao-Jun Michael Shi, Jianyu Huang, Narayanan Sundaraman, Jongsoo Park, Xiaodong Wang, Udit Gupta, Carole-Jean Wu, Alisson G Azzolini, et al. Deep learning recommendation model for personalization and recommendation systems. *arXiv preprint arXiv:1906.00091*, 2019.

[21] Jonathan Frankle, Gintare Karolina Dziugaite, Daniel Roy, and Michael Carbin. Linear mode connectivity and the lottery ticket hypothesis. In *International Conference on Machine Learning*, pages 3259–3269. PMLR, 2020.

[22] Yu Cheng, Felix X Yu, Rogerio S Feris, Sanjiv Kumar, Alok Choudhary, and Shi-Fu Chang. An exploration of parameter redundancy in deep networks with circulant projections. In *Proceedings of the IEEE international conference on computer vision*, pages 2857–2865, 2015.

[23] Vikas Sindhwani, Tara Sainath, and Sanjiv Kumar. Structured transforms for small-footprint deep learning. *Advances in Neural Information Processing Systems*, 28, 2015.

[24] Zichao Yang, Marcin Moczulski, Misha Denil, Nando De Freitas, Alex Smola, Le Song, and Ziyu Wang. Deep fried convnets. In *Proceedings of the IEEE international conference on computer vision*, pages 1476–1483, 2015.

[25] Tri Dao, Beidi Chen, Kaizhao Liang, Jiaming Yang, Zhao Song, Atri Rudra, and Christopher Re. Pixelated butterfly: Simple and efficient sparse training for neural network models. *arXiv preprint arXiv:2112.00029*, 2021.

[26] Amir Gholami, Sehoon Kim, Zhen Dong, Zhewei Yao, Michael W Mahoney, and Kurt Keutzer. A survey of quantization methods for efficient neural network inference. In *Low-Power Computer Vision*, pages 291–326. Chapman and Hall/CRC, 2022.

[27] Aditya Desai and Anshumali Shrivastava. The trade-offs of model size in large recommendation models: 100GB to 10MB Criteo-tb DLRM model. In S. Koyejo, S. Mohamed, A. Agarwal, D. Belgrave, K. Cho, and A. Oh, editors, *Advances in Neural Information Processing Systems*, volume 35, pages 33961–33972. Curran Associates, Inc., 2022.

[28] Ilya Tolstikhin, Neil Houlsby, Alexander Kolesnikov, Lucas Beyer, Xiaohua Zhai, Thomas Unterthiner, Jessica Yung, Andreas Steiner, Daniel Keysers, Jakob Uszkoreit, Mario Lucic, and Alexey Dosovitskiy. Mlp-mixer: An all-mlp architecture for vision, 2021.

[29] Stephen Merity, Caiming Xiong, James Bradbury, and Richard Socher. Pointer sentinel mixture models, 2016.

[30] Alex Krizhevsky, Geoffrey Hinton, et al. Learning multiple layers of features from tiny images. 2009.

[31] Ya Le and Xuan S. Yang. Tiny imagenet visual recognition challenge. 2015.

[32] Guangxuan Xiao, Ji Lin, Mickael Seznec, Hao Wu, Julien Demouth, and Song Han. Smoothquant: Accurate and efficient post-training quantization for large language models. In *International Conference on Machine Learning*, pages 38087–38099. PMLR, 2023.

[33] Thomas Wolf, Lysandre Debut, Victor Sanh, Julien Chaumond, Clement Delangue, Anthony Moi, Pierric Cistac, Tim Rault, Rémi Louf, Morgan Funtowicz, et al. Huggingface's transformers: State-of-the-art natural language processing. *arXiv preprint arXiv:1910.03771*, 2019.

[34] Adam Paszke, Sam Gross, Francisco Massa, Adam Lerer, James Bradbury, Gregory Chanan, Trevor Killeen, Zeming Lin, Natalia Gimelshein, Luca Antiga, Alban Desmaison, Andreas Kopf, Edward Yang, Zachary DeVito, Martin Raison, Alykhan Tejani, Sasank Chilamkurthy, Benoit Steiner, Lu Fang, Junjie Bai, and Soumith Chintala. Pytorch: An imperative style, high-performance deep learning library. In H. Wallach, H. Larochelle, A. Beygelzimer, F. d'Alché-Buc, E. Fox, and R. Garnett, editors, *Advances in Neural Information Processing Systems*, volume 32. Curran Associates, Inc., 2019.

[35] Philippe Tillet, Hsiang-Tsung Kung, and David Cox. Triton: an intermediate language and compiler for tiled neural network computations. In *Proceedings of the 3rd ACM SIGPLAN International Workshop on Machine Learning and Programming Languages*, pages 10–19, 2019.

[36] Paulius Micikevicius, Sharan Narang, Jonah Alben, Gregory Diamos, Erich Elsen, David Garcia, Boris Ginsburg, Michael Houston, Oleksii Kuchaiev, Ganesh Venkatesh, et al. Mixed precision training. *arXiv preprint arXiv:1710.03740*, 2017.

[37] Tri Dao, Dan Fu, Stefano Ermon, Atri Rudra, and Christopher Ré. Flashattention: Fast and memory-efficient exact attention with io-awareness. In S. Koyejo, S. Mohamed, A. Agarwal, D. Belgrave, K. Cho, and A. Oh, editors, *Advances in Neural Information Processing Systems*, volume 35, pages 16344–16359. Curran Associates, Inc., 2022.

[38] Dan Hendrycks, Collin Burns, Steven Basart, Andy Zou, Mantas Mazeika, Dawn Song, and Jacob Steinhardt. Measuring massive multitask language understanding, 2021.

[39] Keisuke Sakaguchi, Ronan Le Bras, Chandra Bhagavatula, and Yejin Choi. Winogrande: An adversarial winograd schema challenge at scale, 2019.

[40] Leo Gao, Jonathan Tow, Baber Abbasi, Stella Biderman, Sid Black, Anthony DiPofi, Charles Foster, Laurence Golding, Jeffrey Hsu, Alain Le Noac'h, Haonan Li, Kyle McDonell, Niklas Muennighoff, Chris Ociepa, Jason Phang, Laria Reynolds, Hailey Schoelkopf, Aviya Skowron, Lintang Sutawika, Eric Tang, Anish Thite, Ben Wang, Kevin Wang, and Andy Zou. A framework for few-shot language model evaluation, 12 2023.

[41] Felipe Maia Polo, Lucas Weber, Leshem Choshen, Yuekai Sun, Gongjun Xu, and Mikhail Yurochkin. tinybenchmarks: evaluating llms with fewer examples, 2024.

# A   Additional Details for checklist

1. **Resource Details:**
   - MLPMixer experiments: Each run takes 2 hours on a single RTX Quadro 8000 machine
   - GPT2-Small experiments: Each run takes around 13 hours on four 32-GB V100 GPUs
   - BERT experiments: QQP, the largest dataset for finetuning with BERT-Large, takes around 7-8 hours on RTX-8000 Quadro.
   - Llama experiments: on MMLU takes 1 hour on 1 40GB A100

2. **License details:**
   - MLPMixer Experiments uses: CIFAR-10 , CIFAR-100 datasets [30] (license: could not find) with MLPMixer GitHub https://github.com/omihub777/MLP-Mixer-CIFAR (MIT License)
   - GPT2 Experiments uses: Wikitext-103 dataset [29] (Creative Commons Attribution-ShareAlike License (CC BY-SA 4.0) License) the code is obtained from fly GitHub repo: https://github.com/HazyResearch/fly presented in [16] (Apache-2.0 license), the quantization experiments on GPT2 uses the quantization method employed in [32] with GitHub repo: https://github.com/mit-han-lab/smoothquant (MIT License) Codes are generally obtained from the HuggingFace Transformers library[33] including GPT2 tokenzier the repo is at https://github.com/huggingface/transformers with Apache-2.0 License. The backend of experiments is Pytorch[34] at https://github.com/pytorch/pytorch
   - Llama3 [1] pre-trained weights and code are obtained from https://github.com/meta-llama/llama3 (META LLAMA 3 COMMUNITY LICENSE AGREEMENT)
   - The kernels are implemented with Triton [35] with GitHub repo: https://github.com/triton-lang/triton (MIT License)
   - Massive Multitask Language Understanding Dataset - MIT License. Obtained via HuggingFace Dataset library.
   - Winogrande Dataset - Apache 2.0 License. Obtained via HuggingFace Dataset library.
   - The Microsoft Research Paraphrase Corpus - License unknown. Obtained via HuggingFace Dataset library.
   - The Multi-Genre Natural Language Inference Corpus - MIT License. Obtained via HuggingFace Dataset library.
   - The Quora Question Pairs2 dataset - Licensed by Quora. https://www.quora.com/about/tos. Obtained via HuggingFace Dataset library.
   - The Corpus of Linguistic Acceptability - Obtained via HuggingFace Dataset library.
   - The Recognizing Textual Entailment (RTE) datasets - Unknown license. Obtained via HuggingFace Dataset library.
   - The Stanford Sentiment Treebank - CC0: Public Domain license. Obtained via HuggingFace Dataset library.
   - The Semantic Textual Similarity Benchmark - Unknown license. Obtained via HuggingFace Dataset library.
   - The Winograd Schema Challenge - Unknown license. Obtained via HuggingFace Dataset library.
   - Question-answering NLI dataset - CC BY-SA 4.0 license. Obtained via HuggingFace Dataset library.

# B   Predicted Common Questions (PCQ)

1. **Q: How do you choose datasets?**
   We begin with what Monarch evaluated w.r.t deep learning – train GPT and finetune BERT. Additionally, we wanted to do a quality tradeoff analysis on wider data. So, we chose the vision datasets that LTH uses. Apart from that, we choose the DLRM MLPerf benchmark as a representative of recommendation models.

2. **Q: Do you see any training latency benefits?**
   The computation in the backward pass is not reduced in SS1. So, in general, the backward

pass costs the same as the full model. In this sense, SS1 is geared towards inference benefits. We might see some improvements due to the lower overall memory footprint and metadata that the optimizer stores. But beyond that, we do not expect to see training latency benefits. Under the assumption that inference is more costly than training, this seems to be an okay thing.

3. **Q: Why have you not measured latency with Quantization?**
   To measure latency with Quantization, we would have to write a quantized SSL kernel. For this paper, we want to restrict our code development to a single kernel. Thus, we only show the quality preservation with SS1 + Quantization. Having said that, we expect the gains from Quantization to be present in SS1 + Quantization.

4. Q: **Does this mean standard linear layers should always be replaced by the SS1 ?**
   We would like to believe there is an "additional" representative value in SS1 layers that can push the boundary of quality per unit compute. However, we want to be cautious while making this claim, and we will be rigorously evaluating SS1 in the future.

5. **Q: Why not block-sparse as a baseline?**
   Most of our experiments were done in FP16. Also, our kernels are optimized for FP16. Pytorch block sparse does not have a stable implementation when it comes to FP16. We see segmentation faults. Thus, we skip Block sparse as a baseline. Additionally, with recent work on sparsity vs. RPS, we believe that RPS should beat Block sparsity in quality per unit parameters.

6. **Q: Monarch performance seems lower than what was reported in the original paper.**
   We were surprised as well. We expected Monarch to do better. However, the paper also spoke only about the training times. In our paper, we measure inference times. Additionally, we confirmed with the authors that we were benchmarking the correct code.

7. **Q: Why was expressivity of SS1 not analyzed the way it has been done for other methods such as Monarch?** While the analysis of Monarch is interesting and a valuable piece of knowledge, it is debatable if the expressivity analysis is representative of deep learning requirements. Indeed, we show that SS1 beats Monarch in quality vs. parameters ( and compute), emphasizing that deep learning may not align with the notion of common transformations.

8. **Q: Will the results on GPT translate to bigger models such as Llama?** It is hard to say. But we believe that whenever the model is at capacity with the information it can learn, having these layers in a slightly bigger model ( so it has a similar computation as a smaller model) can help boost the quality.

9. **Q: Where is the code?** Visit our landing page: https://github.com/apd10/Sketch-Structured-Linear for all the details.

10. **Q: What care needs to be taken for dense-to-structured finetuning?** Different layers have different sensitivity to compression, especially when we want to preserve the accuracy of the pre-trained model. Thus, we have to identify layers that cannot withstand the approximation. This can be done using some calibration data.

11. **Q: Isn't this just feature hashing?** No. Feature hashing (followed by linear transformation) is a low-rank operation, with the first matrix being the fixed sketch matrix. Also, feature hashing is not GPU-aware and has bad latency performance. SS1 applies feature hashing but for each neuron separately. The independence makes SS1 full rank with high probability. Also, due to the coalescing factors, we make SS1 GPU aware and efficient.

## C   How does learning rate tuning affect the quality?

In our experiments, we always use the hyperparameters that are optimal for the original model. However, these hyperparameters are not optimal for different methods. We perform learning rate tuning for Monarch, a competitive baseline, and find that, while the perplexity improves for all methods with finetuning, the findings remain unchanged. The results are shown in 6

Table 6: Learning tuning on Monarch.

| Method | Learning Rate | Loss | PPL |
|---|---|---|---|
| **Monarch-8x** | 6e-4 | 3.119 | 22.83 |
| | 9e-4 | 3.135 | 23.23 |
| | 2e-3 | 3.107 | 20.65 |
| | 8e-3 | NA | NA |
| **SS1-8x** | 2e-3 | 2.99 | 20.13 |
| **LowRank-8x** | 2e-3 | 3.161 | 23.67 |

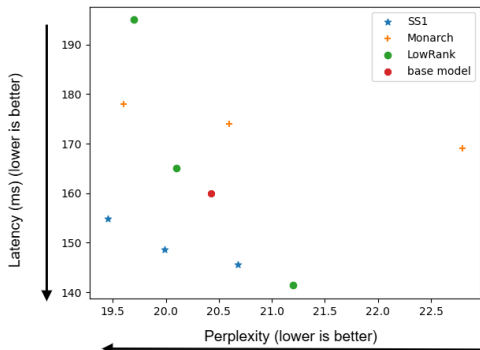

Figure 3: Latency vs. Perplexity Plot for GPT-S

## D  Different representation of latency quality results

The plot of latency vs. quality for GPT-2 is presented in figure 3.

## E  Latency vs. Parameters and Kernel Details

We have implemented three kernels for SS1, including *forward*, *backward-weight-gradient*, and *backward-input-gradient*. Our implementations can be used on GPUs with or without tensor cores and support multiple precisions, such as *float16* and *float32* (TF32).

We implemented permutation by adjusting the addresses to load instead of permuting the data in shared memory. If permuting is done in shared memory, we have to permute the data again to adjust the value distribution to fit the tensor core operands, thus wasting instructions on shared memory accesses and time in synchronizing threads within a CTA (Cooperative Thread Array).

In the *forward* kernel, we launch $\lceil M/\mathrm{B_M} \rceil \times \lceil N/\mathrm{B_N} \rceil$ CTAs. Each CTA, while accumulating along the $K$ dimension, permutes the input block (i.e., $X$) with each data load. In the *backward-input-gradient* kernel, each CTA permutes the weight block while accumulating along the $N$ dimension. In the *backward-weight-gradient* kernel, each CTA only permutes the result before storing it in DRAM using `atomicAdd`.

We have introduced the following optimizations to improve the performance of SS1 kernels:

1. **Vectorized Memory Load**. Rather than permuting individual elements, we permute larger chunks of $VEC$ elements along the $K$ dimension to maximize the GPU's memory bandwidth efficiency. The size of $VEC$ can be selected from 1, 2, 4, or 8.

2. **Pipelining**. Utilizing asynchronous memory load instructions supported by the GPU architecture, we preload $N_{stages}$ blocks prior to the main accumulation loop to overlap compute and memory transactions.

3. **Autotuning**. To optimize performance across different tensor shapes, we implemented an autotuner to select the ideal $\mathrm{B_M}$, $\mathrm{B_N}$, $\mathrm{B_K}$, and $N_{stages}$. We also prune configurations that exceed the GPU's shared memory capacity before compilation and benchmarking.

4. **Fusion**. We have fused bias computation directly into the forward kernel to reduce kernel launch and memory access overhead.

SS1's forward kernel reduces the number of load instructions and computational overhead compared to full matrix multiplications. However, the backward kernels of SS1 maintain the same level of load instructions and computational demand as those of full matrix multiplications, leading to similar performance outcomes.

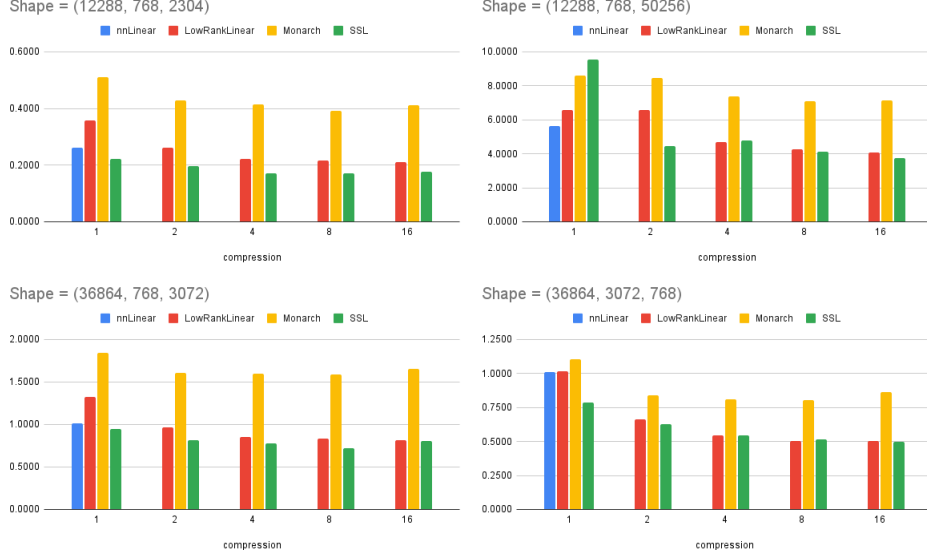

Figure 4: Latency(ms) vs. Parameters for an instance of workload in GPT2

# F  Theoretical Results

## F.1  Claim: $B_K$ is a pure latency parameter

We will evaluate this under the standard dimensionality reduction setup (which we now know is related to learned model quality). Consider two vectors $\mathbf{x}$ and $\mathbf{y}$ in $\mathbb{R}^n$ space. Also, consider the smaller dimension $m$ to which these vectors are being projected using hash functions defined in section 3 with K-coalescing factor $B_K$. Let the final mapping be $h$. The inner product can be written as,

$$\langle \hat{\mathbf{x}}, \hat{\mathbf{y}} \rangle = \sum_{i=1}^{m} \left( \sum_{j=1}^{n} g(j)\mathbf{1}(h(j)=i)x_j \right) \left( \sum_{j=1}^{n} g(j)\mathbf{1}(h(j)=i)y_j \right) \tag{21}$$

Assume that $B_K|m|n$ for simplicity. Then, the inner product can be written as,

$$\langle \hat{\mathbf{x}}, \hat{\mathbf{y}} \rangle = \langle \mathbf{x}, \mathbf{y} \rangle + \sum_{i \neq j}^{n} g(i)g(j)\mathbf{1}(h(i)=h(j))x_i y_j \tag{22}$$

The estimator is unbiased since $E(g(i)) = 0$. The variance is,

$$V(\langle \hat{\mathbf{x}}, \hat{\mathbf{y}} \rangle) = E\left( (\langle \hat{\mathbf{x}}, \hat{\mathbf{y}} \rangle - \langle \mathbf{x}, \mathbf{y} \rangle)^2 \right) = E\left( \sum_{i \neq j}^{n} g(i)g(j)\mathbf{1}(h(i)=h(j))x_i y_j \right)^2 \tag{23}$$

$$V(\langle \hat{\mathbf{x}}, \hat{\mathbf{y}} \rangle) = \tag{24}$$

$$E\left( \sum_{i_1 \neq j_1, i_2 \neq j_2}^{n} g(i_1)g(j_1)g(i_2)g(j_2)\mathbf{1}(h(i_1)=h(j_1))\mathbf{1}(h(i_2)=h(j_2))x_{i_1}y_{j_1}x_{i_2}y_{j_2} \right) \tag{25}$$

Nonzero terms appear if either of the two cases holds:

- $(i_1 = i_2$ and $j_1 = j_2)$
- $(i_1 = j_2$ and $i_2 = j_1)$

$$V(\langle \hat{\mathbf{x}}, \hat{\mathbf{y}} \rangle - \langle x, y \rangle) = E\left( \sum_{i \neq j=1}^{n} \mathbf{1}(h(i){=}h(j)) x_i^2 y_j^2 \right) + E\left( \sum_{i \neq j=1}^{n} \mathbf{1}(h(i){=}h(j)) x_i y_i x_j y_j \right)$$
(26)

$$= \left( \sum_{i \neq j=1}^{n} \mathbf{P}(h(i){=}h(j)) x_i^2 y_j^2 \right) + \left( \sum_{i \neq j=1}^{n} \mathbf{P}(h(i){=}h(j)) x_i y_i x_j y_j \right)$$
(27)

$$= \sum_{i=1}^{n} \sum_{j=1, j \neq i}^{n} \mathbf{P}(h(i){=}h(j)) \left( x_i^2 y_j^2 + x_i y_i x_j y_j \right)$$
(28)

Note that

$$\mathbf{P}(h(i){=}h(j)) = \begin{cases} \frac{1}{B_K} & \text{i,j belong to different groups inside same 1D super group} \\ 0 & \text{otherwise} \end{cases}$$

Let $I(i,j)$ be the delta function that $i,j$ belong to different groups inside same 1D supergroup

$$V(\langle \hat{\mathbf{x}}, \hat{\mathbf{y}} \rangle - \langle \mathbf{x}, \mathbf{y} \rangle) = \frac{1}{B_K} \sum_{i=1}^{n} \sum_{j=1, j \neq i}^{n} I(i,j) \left( x_i^2 y_j^2 + x_i y_i x_j y_j \right)$$
(29)

Under a permutation $p$ of the initial input $x$ and $y$, the following is the probability since every element can interact with all the elements from other groups inside the same supergroup.

$$\mathbf{P}_p(I(i,j)) = \frac{(c-1)B_K}{n-1}$$

Thus, under permutation,

$$\mathbf{E}_p\left( V(\langle \hat{\mathbf{x}}, \hat{\mathbf{y}} \rangle - \langle \mathbf{x}, \mathbf{y} \rangle) \right) = \frac{(c-1)B_K}{(n-1)B_K} \sum_{i=1}^{n} \sum_{j=1, j \neq i}^{n} \left( x_i^2 y_j^2 + x_i y_i x_j y_j \right)$$
(30)

$$\mathbf{E}_p\left( V(\langle \hat{\mathbf{x}}, \hat{\mathbf{y}} \rangle - \langle \mathbf{x}, \mathbf{y} \rangle) \right) = \frac{c-1}{(n-1)} \sum_{i=1}^{n} \sum_{j=1, j \neq i}^{n} \left( x_i^2 y_j^2 + x_i y_i x_j y_j \right)$$
(31)

Thus, $B_K$ is a pure latency parameter and should be chosen for optimizing shared memory usage.

### F.2 How does $B_N$ affect the quality?

To investigate the effect of $B_N$, let us consider how neurons in the same 2D super group and different 2D super groups affect each other. We present the result in the following theorem,

**Theorem 3** *Consider two neurons $y_1, y_2 \in R$ output from a linear transformation under SS1. Let $h_1, h_2, g_1, g_2$ be the hash functions used for the neurons. Let the weights for these two neurons be $p, q \in R^m$. Let the input be $x \in R^n$. Then, under the assumption that each signal in $x \sim \mathcal{D}$ is independent, the Covariance between the values of neurons can be written as,*

$$Cov_{\mathcal{D}}(y_1, y_2) = \sum_{i_1=1}^{m} \sum_{i_1=1}^{m} \sum_{j=1}^{n} p_{i_1} q_{i_2} g_1(j) g_2(j) \mathbf{1}(h_1(j){=}i_1) \mathbf{1}(h_2(j){=}i_2) \sigma_{x_j}^2$$
(32)

$$\mathbf{E}_{h_1, g_1, h_2, g_2}(Cov_{\mathcal{D}}(y_1, y_2)) = \begin{cases} \frac{m}{n} ||\sigma_x||^2 \langle p, q \rangle & \text{if } h_1 = h_2, g_1 = g_2 \\ 0 & \text{if } g_1 \neq g_2 \end{cases}$$
(33)

The above theorem points out that if two neurons belong to the same group, i.e., they share the hash functions, then the neuron outputs have unintended correlations and, thus, mutual information. So the larger value of $B_N$ implies more neurons having interdependency and, consequently, lesser expressivity. Fortunately, in our experiments, we find that larger values of $B_N$ also work well, and we optimize it for latency.

**details:** Intuitively, sharing the mapping functions across neurons will restrict the representative power of neurons since they will sketch the same parts of the input vector $x$ with the same hash functions. We verify this intuition here. Consider the following two neurons under SS1 with weight vectors $p, q$

$$y_1 = \sum_{i=1}^{m} p_i \left( \sum_{j=1}^{n} g_1(j) \mathbf{1}(h_1(j){=}i) x_j \right)$$

$$y_2 = \sum_{i=1}^{m} q_i \left( \sum_{j=1}^{n} g_2(j) \mathbf{1}(h_2(j){=}i) x_j \right)$$

Consider the mutual information between $y_1$ and $y_2$. Since we are dealing with linear computations, we measure the Covariance of the two neurons,

$$Cov_{\mathcal{D}}(y_1, y_2) = Cov \left( \sum_{i=1}^{m} p_i \left( \sum_{j=1}^{n} g_1(j) \mathbf{1}(h_1(j){=}i) x_j \right), \sum_{i=1}^{m} q_i \left( \sum_{j=1}^{n} g_2(j) \mathbf{1}(h_2(j){=}i) x_j \right) \right) \tag{34}$$

$$Cov_{\mathcal{D}}(y_1, y_2) = \sum_{i_1=1}^{m} \sum_{j_1=1}^{n} \sum_{i_1=1}^{m} \sum_{j_1=1}^{n} Cov \left( p_{i_1} \left( g_1(j_1) \mathbf{1}(h_1(j_1){=}i_1) x_{j_1} \right), q_{i_2} \left( g_2(j_2) \mathbf{1}(h_2(j_2){=}i_2) x_{j_2} \right) \right) \tag{35}$$

$$Cov_{\mathcal{D}}(y_1, y_2) = \sum_{i_1=1}^{m} \sum_{j_1=1}^{n} \sum_{i_1=1}^{m} \sum_{j_1=1}^{n} Cov \left( p_{i_1} \left( g_1(j_1) \mathbf{1}(h_1(j_1){=}i_1) x_{j_1} \right), q_{i_2} \left( g_2(j_2) \mathbf{1}(h_2(j_2){=}i_2) x_{j_2} \right) \right) \tag{36}$$

$$Cov_{\mathcal{D}}(y_1, y_2) = \sum_{i_1=1}^{m} \sum_{j_1=1}^{n} \sum_{i_1=1}^{m} \sum_{j_1=1}^{n} p_{i_1} q_{i_2} g_1(j_1) g_2(j_2) \mathbf{1}(h_1(j_1){=}i_1) \mathbf{1}(h_2(j_2){=}i_2) Cov(x_{j_1}, x_{j_2}) \tag{37}$$

Assuming independent signals in $x$

$$Cov_{\mathcal{D}}(y_1, y_2) = \sum_{i_1=1}^{m} \sum_{i_1=1}^{m} \sum_{j=1}^{n} p_{i_1} q_{i_2} g_1(j) g_2(j) \mathbf{1}(h_1(j){=}i_1) \mathbf{1}(h_2(j){=}i_2) \sigma_j^2 \tag{38}$$

**Case 1: $y_1$ and $y_2$ belong to the same 2D super group**

$$\mathbf{E}_{h,g}(Cov_{\mathcal{D}}(y_1, y_2) = \left( \sum_{i_1=1}^{m} \sum_{i_1=1}^{m} \sum_{j=1}^{n} p_{i_1} q_{i_2} g(j)^2 \mathbf{1}(h(j){=}i_1) \mathbf{1}(h(j){=}i_2) \sigma_j^2 \right) \tag{39}$$

$$\mathbf{E}_{h,g}(Cov_{\mathcal{D}}(y_1, y_2) = \left( \sum_{i_1=1}^{m} \sum_{i_2=1}^{m} \sum_{j=1}^{n} p_{i_1} q_{i_2} E(\mathbf{1}(h(j){=}i_1) \mathbf{1}(h(j){=}i_2)) \sigma_j^2 \right) \tag{40}$$

Only non-zero terms when $i_1 = i_2$ since $h(j)$ can only have one value

$$\mathbf{E}_{h,g}(Cov_{\mathcal{D}}(y_1, y_2)) = \left( \sum_{i=1}^{m} \sum_{j=1}^{n} p_i q_i E(\mathbf{1}(h(j){=}i)) \sigma_j^2 \right) \tag{41}$$

$$\mathbf{E}_{h,g}(Cov_{\mathcal{D}}(y_1, y_2)) = \frac{m}{n} ||\sigma||^2 \langle p, q \rangle \tag{42}$$

**Case 2: $y_1$ and $y_2$ belong to the different 2D super group**
$$\mathbf{E}_{h,g}(Cov_{\mathcal{D}}(y_1, y_2)) = 0 \tag{43}$$

Thus, if two neurons are in the same 2D group, the structure imposes the two neurons to be correlated and thus reduces the expressive power of the model.

### F.3 Quantization + SS1: How combining them results in a better method for compression.

**Analysing quantization under standard dimensionality reduction**    Consider two vectors $x, y$ with values from the range $[-D/2, D/2]$. Under stochastic Quantization for b bits, we will have
$$\hat{\mathbf{x}} = x + \Delta \tag{44}$$
where $\Delta[i] = (\lceil x \rceil - x)\mathbf{1}(i) + (\lfloor x \rfloor - x)(1 - \mathbf{1}(i))$ where
$$\mathbf{1}(i) = \begin{cases} 1 \text{ with prob } p = \frac{(x - \lfloor x \rfloor)}{D/2^b} \\ 0 \text{ with prob } (1 - p) \end{cases} \tag{45}$$

the inner product
$$\langle \hat{\mathbf{x}}, \hat{\mathbf{y}} \rangle = \langle \mathbf{x}, \mathbf{y} \rangle + \langle x, \Delta_y \rangle + \langle \Delta_x, y \rangle + \langle \Delta_x, \Delta_y \rangle \tag{46}$$
The estimator is unbiased.

$$\langle \hat{\mathbf{x}}, \hat{\mathbf{y}} \rangle - \langle \mathbf{x}, \mathbf{y} \rangle = \sum_{j=1}^{n} (x_j \Delta_y[j] + y_j \Delta_x[j] + \Delta_x[j]\Delta_y[j]) \tag{47}$$

$$V(\langle \hat{\mathbf{x}}, \hat{\mathbf{y}} \rangle) = \sum_{j=1}^{n} V(x_j \Delta_y[j] + y_j \Delta_x[j] + \Delta_x[j]\Delta_y[j]) \tag{48}$$

$$V(\langle \hat{\mathbf{x}}, \hat{\mathbf{y}} \rangle) = \sum_{j=1}^{n} Cov(x_j \Delta_y[j] + y_j \Delta_x[j] + \Delta_x[j]\Delta_y[j], x_j \Delta_y[j] + y_j \Delta_x[j] + \Delta_x[j]\Delta_y[j]) \tag{49}$$

$$V(\langle \hat{\mathbf{x}}, \hat{\mathbf{y}} \rangle) = \sum_{j=1}^{n} V(x_j \Delta_y[j]) + V(y_j \Delta_x[j]) + V(\Delta_x[j]\Delta_y[j]) \tag{50}$$

$$+ Cov(x_j \Delta_y[j], \Delta_x[j]\Delta_y[j]) + Cov(y_j \Delta_x[j], \Delta_x[j]\Delta_y[j]) \tag{51}$$

Convince yourself that covariance terms are 0.

$$V(\langle \hat{\mathbf{x}}, \hat{\mathbf{y}} \rangle) = \sum_{j=1}^{n} V(x_j \Delta_y[j]) + V(y_j \Delta_x[j]) + V(\Delta_x[j]\Delta_y[j]) \tag{52}$$

$$V(\langle \hat{\mathbf{x}}, \hat{\mathbf{y}} \rangle) = \sum_{j=1}^{n} x_j^2 \sigma_{yj}^2 + y_j^2 \sigma_{xj}^2 + \sigma_{yj}^2 \sigma_{xj}^2 \tag{53}$$

$$\sigma_{xj}^2 = V((\lceil x \rceil - x)\mathbf{1}(i) + (\lfloor x \rfloor - x)(1 - \mathbf{1}(i))) \tag{54}$$

$$\sigma_{xj}^2 = \left(\frac{D}{2^b}\right)^2 V(\mathbf{1}(j)) = \left(\frac{D}{2^b}\right)^2 \frac{(x - \lceil x \rceil)}{D/2^b} \frac{(D/2^b - (x - \lceil x \rceil))}{D/2^b} \tag{55}$$

$$\sigma_{xj}^2 = (x - \lfloor x \rfloor)(D/2^b - (x - \lfloor x \rfloor)) = w^2 p_{xj}(1 - p_{xj}) \leq \frac{w^2}{4} \tag{56}$$

Note that the inequality is tight and occurs when all the values of $x$ are exactly at the midpoint of the bin.

where $w = D/2^b$

$$V_q(\langle \hat{\mathbf{x}}, \hat{\mathbf{y}} \rangle) = \sum_{j=1}^{n} x_j^2 \sigma_{yj}^2 + y_j^2 \sigma_{xj}^2 + \sigma_{yj}^2 \sigma_{xj}^2 \leq_t \frac{w^2}{4} \left( ||x||^2 + ||y||^2 \right) + n \frac{w^4}{16} \tag{57}$$

It follows that this inequality is also tight.

The equation for projection is

$$V_p(\langle \hat{\mathbf{x}}, \hat{\mathbf{y}} \rangle) = \frac{1}{m} \sum_{i \neq j} (x_i^2 y_j^2 + x_i y_i x_j y_j) \tag{58}$$

$$V_p(\langle \hat{\mathbf{x}}, \hat{\mathbf{y}} \rangle) = \frac{1}{m} \left( ||x||^2 ||y||^2 + \langle x, y \rangle^2 - 2||x \circ y||^2 \right) \tag{59}$$

if $||x|| \leq 1, ||y|| \leq 1$

$$V_q(\langle \hat{\mathbf{x}}, \hat{\mathbf{y}} \rangle) \leq \frac{w^2}{2} + \frac{nw^4}{16} \leq \frac{2}{2^{2b}} + \frac{n}{2^{4b}} = Q(b) \tag{60}$$

$$V_p(\langle \hat{\mathbf{x}}, \hat{\mathbf{y}} \rangle) \leq_t \frac{k}{m} = P(m) \leq \frac{2}{m} \tag{61}$$

we use $\leq_t$ to show that it is tight. The tight upper bound can be represented using some $1 \leq k \leq 2$ Under same budget $m = \frac{nb}{F}$ where $F = 32$ or $F = 16$. for whatever precision width we are using for the original vectors.

$$V_p(\langle \hat{\mathbf{x}}, \hat{\mathbf{y}} \rangle) \leq \frac{kF}{nb} \tag{62}$$

### F.3.1  Combined

Let us consider $x$ and $y$ with norm 1. Let us consider the procedure where we first apply projections to get $\hat{\mathbf{x}}_1$ and $\hat{\mathbf{y}}_1$, and then Quantization is applied to obtain $\hat{\mathbf{x}}_2$ and $\hat{\mathbf{y}}_2$. Let us assume that the bit width for Quantization is $b$ and the projection is into $m$.

Let's apply Quantization and then projection. The resulting estimator can be written as

$$\langle \hat{\mathbf{x}}, \hat{\mathbf{y}} \rangle = \langle x + \Delta_x, y + \Delta_y \rangle + \sum_{i \neq j}^{n} g(i)g(j)\mathbf{1}(h(i) == h(j))(x_i + \Delta_x[i])(y_j + \Delta_y[j]) \tag{63}$$

The estimator is unbiased. Now let us look at the variance w.r.t all stochastic elements.

$$\langle \hat{\mathbf{x}}, \hat{\mathbf{y}} \rangle = \langle x, y \rangle + \langle x, \Delta_y \rangle + \langle \Delta_x, y \rangle + \langle \Delta_x, \Delta_y \rangle + \sum_{i \neq j}^{n} g(i)g(j)\mathbf{1}(h(i) == h(j))(x_i + \Delta_x[i])(y_j + \Delta_y[j]) \tag{64}$$

$$\langle \hat{\mathbf{x}}, \hat{\mathbf{y}} \rangle - \langle x, y \rangle = \langle x, \Delta_y \rangle + \langle \Delta_x, y \rangle + \langle \Delta_x, \Delta_y \rangle + \sum_{i \neq j}^{n} g(i)g(j)\mathbf{1}(h(i) == h(j))(x_i + \Delta_x[i])(y_j + \Delta_y[j]) \tag{65}$$

Note that each of the term on the right side has covariance 0 with each other since $E(AB) = E(A)E(B) = 0$, so $Cov(A, B) = 0$ Hence,

$$V(\langle \hat{\mathbf{x}}, \hat{\mathbf{y}} \rangle) = V(\langle x, \Delta_y \rangle + \langle \Delta_x, y \rangle + \langle \Delta_x, \Delta_y \rangle) + V(\sum_{i \neq j}^{n} g(i)g(j)\mathbf{1}(h(i) == h(j))(x_i + \Delta_x[i])(y_j + \Delta_y[j]))$$
(66)

$$V(\langle \hat{\mathbf{x}}, \hat{\mathbf{y}} \rangle) = V_q(x, y) + \tag{67}$$

$$E_q \left( \frac{1}{m} \sum_{i \neq j} ((x_i + \Delta_x[i])^2 (y_j + \Delta_y[i])^2 + (x_i + \Delta_x[i])(y_i + \Delta_y[i])(x_j + \Delta_x[j])(y_j + \Delta_y[j])) \right)$$
(68)

$$V(\langle \hat{\mathbf{x}}, \hat{\mathbf{y}} \rangle) = V_q(x, y) + \left( \frac{1}{m} \sum_{i \neq j} ((x_i^2 + \sigma_{xi}^2)(y_j^2 + \sigma_{yj}^2) + x_i y_i x_j y_j) \right) \tag{69}$$

$$V(\langle \hat{\mathbf{x}}, \hat{\mathbf{y}} \rangle) = V_q(x, y) + V_p(x, y) + \left( \frac{1}{m} \sum_{i \neq j} ((x_i^2 \sigma_{yj}^2 + y_j^2 \sigma_{xi}^2 + \sigma_{yj}^2 \sigma_{xi}^2) \right) \tag{70}$$

$$\left( \frac{1}{m} \sum_{i \neq j} ((x_i^2 \sigma_{yj}^2 + y_j^2 \sigma_{xi}^2 + \sigma_{yj}^2 \sigma_{xi}^2) \right) \leq \frac{1}{m} \frac{w^2}{4} (2(n-1)) + \frac{1}{m} \frac{n(n-1)}{1} \frac{w^4}{16} \tag{71}$$

Thus,

$$V_{pq}(\langle \hat{\mathbf{x}}, \hat{\mathbf{y}} \rangle) \leq \frac{w^2}{2} + \frac{nw^4}{16} + \frac{k}{m} + \frac{1}{m} \frac{w^2}{2} ((n-1)) + \frac{1}{m} \frac{n(n-1)}{1} \frac{w^4}{16} \tag{72}$$

where $w = D/2^b = 2/2^b$

$$V_{pq}(\langle \hat{\mathbf{x}}, \hat{\mathbf{y}} \rangle) \leq Q(b) \left( 1 + \frac{n-1}{m} \right) + P(m) \tag{73}$$

effective bits per element is $B = (mb)/n = F(m/n)(b/F)) = c_p c_q F$.

$$V_{pq}(\langle \hat{\mathbf{x}}, \hat{\mathbf{y}} \rangle) \leq Q(b) \left( 1 + \frac{n-1}{m} \right) + P(m) \approx Q(c_q)(1 + \frac{1}{c_p}) + P(c_p) \tag{74}$$

$$Q(c_q) = \frac{2}{2^{2b}} + \frac{n}{2^{4b}} = \frac{2}{2^{2c_q F}} + \frac{n}{2^{4c_q F}} \tag{75}$$

$$Q(c_q) = \frac{2}{2^{2b}} + \frac{n}{2^{4b}} = \frac{2}{2^{2c_q F}} + \frac{n}{2^{4c_q F}} \tag{76}$$

$$P(c_p) = \frac{k}{c_p n} \tag{77}$$

$$R(c_p c_q) = \tilde{R}(c_p, c_q) = Q(c_q) \left( 1 + \frac{1}{c_p} \right) + P(c_p) \tag{78}$$

## G  Projection Details

Firstly, note that parameter sharing in SS1 is only inside a single neuron. So, there are independent RPS instances for each neuron. Thus, the projection of the $K \times N$ weight matrix is independently projecting each neuron. Consider a single neuron under RPS $y^\top = x^\top w$ and $w = Sz$. Given $w$, the solution to get best $z$ which minimizes $||w - Sz||_2$ is nothing but solution of linear regression, i.e. $z = (S^\top S)^{-1}S^\top w$. Also, since S is a sparse matrix with exactly one non-zero in each row, $S^\top S$ is a diagonal matrix. The hash function defined in the paper ensures that all elements of the diagonal are non-zero. Thus, it is invertible. In fact, if you view S as a mapping matrix that maps each element of $w$ (i.e., range K) to some element of $z$ ( range $K/c$), then the value of z is just an aggregation of all those elements from $w$ that map to this element in $z$ ( computed via $S^\top w$) and then normalized by the total number of elements from $w$ that map to this element in $z$ ( computed via $S^\top S)^{-1}$ ). This is straightforward to implement and can be done in a blocked manner for the entire matrix $Z$. This is presented in the Algo 2.

---

**Algorithm 2** SS1-project(W)

---

**Require:** $c \in \mathbb{N}$ : compression factor, $\mathbf{W} \in \mathbb{R}^{K \times N}$ : full weight matrix, $\mathrm{B_M}, \mathrm{B_K}, \mathrm{B_N}$: coalescing parameters,$h : \mathbb{N}^3 \rightarrow \{0, \ldots, B_K - 1\}, g : \mathbb{N}^3 \rightarrow \{\pm 1\}, c\mathrm{B_K}|K$
**Ensure:** $\mathbf{Z}^* = \mathrm{argmin}_Z ||\mathbf{W} - SS1(\mathbf{Z}, I(K))||_F$
  1: $\mathbf{Z}^*$=Zero-Matrix($K//c, N$)
  2: $\mathbf{T}_w$=TILE($\mathrm{B_K}, \mathrm{B_N}$)
  3: **for** $j \in [[\lceil N/\mathbf{B_N} \rceil]]$ **do**
  4:      **for** $k \in [[\lceil K//c \rceil]]$ **do**
  5:          $\mathbf{T}_w[:,:] = 0$
  6:          **for** $l \in [c]$ **do**
  7:              $a$=$\mathbf{T}_w[(kc+l)\mathrm{B_K}:(kc+l+1)\mathrm{B_K},$
                   $j\mathrm{B_N}:(j+1)\mathrm{B_N}]$
  8:              $a$=$a[(\mathrm{RG}(\mathrm{B_K})-h(j,k,l))\%\mathrm{B_K},:]$
  9:              $\mathbf{T}_w$+=$g(j,k,l)a$
10:          **end for**
11:          $\mathbf{T}_w$=$\mathbf{T}_w/c$
12:          $\mathbf{Z}^*[k\mathrm{B_K}:(k+1)\mathrm{B_K}, j\mathrm{B_N}:(j+1)\mathrm{B_N}]$=$\mathbf{T}_w$
13:      **end for**
14: **end for**

---

## H  Backward kernel algorithms

We implement both the forward and backward kernels with Triton [35]. The backward pass computes the compressed weight gradients and also the input gradients with respect to the hash functions used in the forward pass. It includes two kernels to output the SS1 weight gradients and input gradients.

The algorithms used in the backward kernels are presented in Algo 3 and Algo 4.

---
**Algorithm 3** SS1-backward: weight grad
---
**Require:** $c \in \mathbf{N}$ : compression factor
**Require:** $O \in \mathbb{R}^{M \times N}$ : output gradient
**Require:** $X \in \mathbb{R}^{M \times N}$ : data matrix
**Require:** $B_M, B_K, B_N$: coalescing parameters
**Require:** $h : \mathbb{N}^3 \to \{0, \dots, B_K - 1\}$
**Require:** $g : \mathbb{N}^3 \to \{\pm 1\}$
**Require:** $cB_K | K$
**Ensure:** $W_g \in \mathbb{R}^{N \times (K//c)}$
1: $w$=TILE$(B_N, B_K)$
2: **for** $m \in [\lceil K/\mathbf{B_K} \rceil]$ **do**
3:      **for** $j \in [\lceil N/\mathbf{B_N} \rceil]$ **do**
4:          $w[:,:] = 0$
5:          **for** $i \in [\lceil M/\mathbf{B_M} \rceil]$ **do**
6:              $w = w + \mathbf{MM}(O^T[jB_N:(j+1)B_N, iB_M:(i+1)B_M],$
                       $X[iB_M:(i+1)B_M, mB_K:(m+1)B_K])$
7:          **end for**
8:          $k = m//c$                            ▷ Index of block in SS1matrix
9:          $l = k\%c$                    ▷ Offset of block in full matrix used for hashing
10:         $w$=$w/c$                ▷ Scaling down gradients by compression factor
11:         $W_g[jB_N:(j+1)B_N, kB_k+(\mathbf{RG}(B_K) - h(j,k,l))\%B_K] =$
                   $g(j,k,l)w + W_g[:, kB_K:(k+1)B_K]$
12:      **end for**
13: **end for**
---

---
**Algorithm 4** SS1-backward: input grad
---
**Require:** $c \in \mathbf{N}$ : compression factor
**Require:** $O \in \mathbb{R}^{M \times N}$ : output gradient
**Require:** $W \in \mathbb{R}^{K//c \times N}$ : SS1matrix
**Require:** $B_M, B_K, B_N$: coalescing parameters
**Require:** $h : \mathbb{N}^3 \to \{0, \dots, B_K - 1\}$
**Require:** $g : \mathbb{N}^3 \to \{\pm 1\}$
**Require:** $cB_K | K$
**Ensure:** $X_g \in \mathbb{R}^{M \times K}$
1: $x$=TILE$(B_M, B_K)$
2: **for** $l \in [c]$ **do**
3:      **for** $k \in [\lceil K//c/\mathbf{B_K} \rceil]$ **do**
4:          **for** $i \in [\lceil M/\mathbf{B_M} \rceil]$ **do**
5:              $x[:,:] = 0$
6:              **for** $j \in [\lceil N/\mathbf{B_N} \rceil]$ **do**
7:                  $x = x + \mathbf{MM}(O[iB_M:(i+1)B_M, jB_N:(j+1)B_N],$
                     $g(j,k,l)W^T[jB_N:(j+1)B_N, kB_k+(\mathbf{RG}(B_K)-h(j,k,l))\%B_K])$
8:              **end for**
9:              $X_g[iB_M:(i+1)B_M, (k+l)B_K + :(k+l+1)B_K]=x$
10:          **end for**
11:      **end for**
12: **end for**
---

## I Experiment settings

### I.1 GPT experiment settings

We follow the standard GPT2-Small implementation of monarch paper [16] from the HazyResearch group at fly repo (https://github.com/HazyResearch/fly.git).

To optimize training the GPT model on our GPUs, we replaced Huggingface's transformers[33] 'Conv1D' layers with PyTorch[34] 'nn.Linear' in the feedforward blocks, as the transformation

remains the same. The architecture of the GPT2 models used in these experiments is detailed in 7. All models are trained using mixed-precision-training [36] with FP16 precision and implemented using PyTorch [34] AMP. We use V100-32GB GPUs to train the models from scratch.

Table 7

| Architecture | #Layers | #Heads | Dimension |
|---|---|---|---|
| GPT2 Small | 12 | 12 | 768 |
| GPT2 Medium | 24 | 16 | 1024 |
| GPT2 Large | 36 | 20 | 1280 |

**Hyperparameters:** For training we use the $AdamW$ optimizer with $\alpha$=6e$-$4, $\beta_1$=0.9, $\beta_2$=0.999, $\epsilon = 1e - 08$ and $weight\_decay = 0.1$ we employ a linear schedule and wramup 1% of steps. The effective batch size is $512$, which is not achievable on our hardware memory; thus, we perform gradient accumulation every 32 step to reach that. All models are trained for 100 epochs. The hyperparameters are adopted from the Monarch paper [16].

**Dataset:** We use wikitext-103 [29] to train, evaluate, and test the model. The reported perplexity numbers are based on an evaluation of the test dataset. The length of the generated sequence is 1024.

### I.1.1  Baselines

**SS1:**   To build SS1 models, we substitute the transformer's feedforward linear layers with our SS1 layer. We use a fixed block size of Block_Size_K = 32, Block_Size_N = 32, and Block_Size_M = 64 for both the forward and backward kernels across all layers during training and testing. Additionally, each SS1 layer is assigned a different seed to ensure reproducibility of the hash functions used. In addition to the coalescing parameters, the vectorization parameter E $VEC$ is set to 4.

**Monarch:** : For Monarch baseline, we convert the same layers as SS1 from the original Monarch implementation [16] and change the number of blocks to achieve different compression rates.

**Lowrank:**  To implement the Lowrank layer, we take the linear layer matrix and parameterize it into two consecutive smaller matrices (linear layers) with reduced intermediate dimensions. The intermediate dimension in the replaced layer will be:

$$K_{lr} = (K_s \times N_s)//(K_s + N_s) \times c$$

Where $K_s$ and $N_s$ are the input and output channels of the original matrix respectively, $c \in \mathbb{N}$ is the compression factor, and $K_{lr}$ is the intermediate dimension of Lowrank matrices. Afterward, like the other baselines, we swapped the linear layers of feedforward and replaced them with the Lowrank layer.

**Smallmodel:** The Smallmodel baseline aims to assess if the mlp layers of feedforward blocks of Large Language Models (LLM) are overparameterized and the impact on model quality by choosing smaller matrices for those layers. To achieve this, we take the same modules of the standard model and adjust the inner_dimension of the linear layers by a compression factor c where $c \in \mathbb{N}$.
We were surprised to find that the smaller version of the model had moderate accuracy degradation. We also experimented with combining the small model and SS1 by training the small model with SS1 matrices for the FFN layers. We observed improved performance with the same number of parameters for 'Smallmodel4x+SS12x' compared to 'Smallmodel8x'. The conversion process is the same as what we did in SS1-conversion.

**BlockSparse:**  A pruning baseline that we sought to conduct the same experiment for was pytorch block sparse[5]. However, pytorch block sparse is constrained to fp32 precision training, and for the large models such as GPT2, it does not fit our available GPU memory space.

FlashAttention [37] suggests a significant latency of the inference phase lies in applying the softmax function on attention scores, which is a memory-bound operation. Using FlashAttention kernels shifts the workload from memory-bound to compute-bound; we know from the [37] that linear transformations are compute-bound because of the extensive matrix multiply operations. Considering all that, the performance contribution of SS1 is in the compute-bound setting because of reducing

the inner dimension of the linear matrices. Thus, in GPT2 latency measurement experiments, we use the Pytorch kernel abstraction of FlashAttention, the scaled-dot-product-attention (SDPA). As the FlashAttention kernels are yet to be released for older GPUs such as V100, we were not able to apply them in the training phase on V100 GPUs; thus, we only replaced them in the latency measurement experiments for all the baselines above. For the latency measurement, we employ a single A100-40GB GPU.

### I.1.2 Quantization

To unravel the potential of SS1 in the quantization domain, we utilize uniform post-training Quantization. The method is as follows:

$$S = \max_i |X_i|$$

$$\hat{S} = (\frac{S}{2^n - 1})$$

$$\hat{X} = \left\lfloor \frac{X}{\hat{S}} \right\rceil$$

Where $X$ is a floating point tensor, and $\hat{X}$ is its quantized version to an integer with $n$ number of bits. $S$ is the scale and $\hat{S}$ is the quantization level. $\lfloor \cdot \rceil$ denotes rounding to the nearest integer. We have chosen the scale to be $\max(|X|)$ as suggested by [32] due to its effectiveness in mitigating the impact of large activations in LLMs.

In our application, we implement the proposed method to reduce the precision of activation and weight tensors from 16 to 8 bits during the inference mode. For weight quantization, we apply the per-channel activation, meaning the scale is computed based on values of the tensor for each output channel; for activation quantization, we do per-token Quantization based on [32]. The quantization process is dynamic in our case. The results are obtained from testing the pre-trained GPT2-standard and SS1 models that we trained.

### I.2 BERT finetuning settings

We conducted experiments to show that pre-trained linear layers can be projected onto SS1 layers while retaining previous knowledge. To validate this, we project the encoder layers of BERT-base and BERT-large models [19] onto SS1. BERT (Bidirectional Encoder Representations from Transformers) consists of a stack of Transformer encoder layers, each containing a self-attention mechanism and a feedforward neural network. In this work, we focus on two BERT model sizes: BERT-base (12 encoder layers, 768 hidden size, 12 attention heads, 110M parameters) and BERT-large (24 encoder layers, 1024 hidden size, 16 attention heads, 340M parameters). We then finetune these models on the GLUE (General Language Understanding Evaluation) benchmark [18]. Hyperparameters used during the finetuning stage are provided in Tables 10 and 11. GLUE consists of a collection of diverse natural language understanding tasks, such as textual entailment, question answering, and sentiment analysis. For our experiments, we utilize the GLUE dataset provided by the Hugging Face Datasets library.

The results are presented in Tables 8 and 9. We are able to achieve 50% compression of the encoder layers on both BERT-base and BERT-large models, with only a minimal drop in model quality across the GLUE tasks. This reduction in parameters is not achieved by uniformly compressing all layers in the encoder. Instead, we only project layers 1, 6, 8, 9, 15, 16, 17, 18, 19, 20, 21, 22, 23, 24 (14 out of 24) for BERT-large and layers 1, 7, 8, 9, 10, 11, 12 (7 out of 12) for BERT-base at 8x compression. This leads to a net compression ratio of $1 - \frac{14/8+10}{12} = 0.51$ for BERT-large and $1 - \frac{7/8+5}{12} = 0.51$ for BERT-base, approximately 50% compression for both models.

These specific layers are selected using the RTE (Recognizing Textual Entailment) task from GLUE as calibration data. We treat RTE as a development set to determine which layers are less sensitive to compression. Layer sensitivity is measured by the degree of performance degradation on RTE when applying SS1 compression to individual layers in the model. The layers chosen by this method are able to generalize well to all other GLUE tasks, achieving evaluation metrics that closely match those of the full-sized BERT models with linear layers.

In addition to parameter reduction, BERT models with SS1 layers are able to achieve up to 1.31x increase in inference throughput. Detailed latency measurements are provided in Table 12. On the GLUE task, our BERT-large model with SS1 compression attains an accuracy of 79.76 (2.6 drop compared to the full model), while our compressed BERT-base model reaches 79.9 accuracy (2.2 drop) - demonstrating that our SS1 projection method can maintain strong performance with significantly reduced parameters and runtime.

Table 8: Fine-tuning results for Bert Large

|  | BERT-L | BERT-L × SS1 |
|---|---|---|
| **#param** | 335M | 181M |
| **COLA** | 69 | 70.8 |
| **STSB** | 89.3 | 87.1 |
| **RTE** | 76.1 | 65.9 |
| **MRPC** | 88.5 | 87.1 |
| **WNLI** | 56.3 | 56.3 |
| **QNLI** | 92 | 88 |
| **QQP** | 90 | 88.3 |
| **SST2** | 92.9 | 90.8 |
| **MNLI** | 86 | 81.9 |
| **Average** | 82.2 | 79.6 ± 0.203 |

Table 9: Fine-tuning results for Bert Base

|  | BERT-L | BERT-L × SS1 |
|---|---|---|
| **#param** | 110M | 66M |
| **COLA** | 83.6 | 76.2 |
| **STSB** | 88.8 | 87.5 |
| **RTE** | 67.5 | 67.5 |
| **MRPC** | 89.9 | 87.7 |
| **WNLI** | 53.5 | 56.3 |
| **QNLI** | 90.2 | 86.5 |
| **QQP** | 89.2 | 88.5 |
| **SST2** | 92.4 | 89.5 |
| **MNLI** | 84.4 | 80.5 |
| **Average** | 82.1 | 79.9 ± 0.066 |

Table 10: Hyper parameters for Bert(SS1) Large

| Task | Batch Size | Learning rate | Epochs |
|---|---|---|---|
| cola | 16 | 2.5e-05 | 5 |
| mnli | 16 | 1e-05 | 3 |
| qnli | 16 | 1e-05 | 3 |
| sst2 | 32 | 1.5e-05 | 3 |
| wnli | 8 | 2e-05 | 5 |
| stsb | 32 | 1.5e-05 | 3 |
| rte | 8 | 1e-05 | 7 |
| mrpc | 16 | 1.5e-05 | 5 |
| qqp | 32 | 1e-05 | 3 |

Table 11: Hyper parameters for Bert(SS1) Base

| Task | Batch Size | Learning rate | Epochs |
|---|---|---|---|
| cola | 16 | 2.5e-05 | 5 |
| mnli | 16 | 1e-05 | 3 |
| qnli | 16 | 2.5e-05 | 3 |
| sst2 | 16 | 1.5e-05 | 3 |
| wnli | 32 | 1.5e-05 | 5 |
| stsb | 32 | 2.5e-05 | 3 |
| rte | 16 | 2.5e-05 | 7 |
| mrpc | 32 | 2.5e-05 | 5 |
| qqp | 16 | 2.5e-05 | 3 |

Table 12: Latency results for Bert Models

| Latency ms (median) | | batch | | | |
|---|---|---|---|---|---|
| Model | 8 | 16 | 32 | 64 | 128 |
| BERT-Large | 12.68755198 | 20.73288059 | 40.77404976 | 79.60927963 | 158.8201599 |
| BERT-Large × SS1 | 18.04614353 | 17.61041546 | 31.79167938 | 60.99251175 | 120.5512314 |
| Throughput Increase for SS1 | 0.7030616792 | 1.177307863 | 1.282538405 | 1.305230386 | 1.317449503 |

## I.3   Llama experiments

Meta's Llama family has emerged as one of the most powerful open-source Large Language Models. They are designed to be powerful and efficient models for various natural language processing tasks. In this work, we focus on the Llama3-8B model, which has 32 decoder layers, a hidden size of 4096, and 32 attention heads, resulting in approximately 8 billion parameters.

We conduct experiments to demonstrate that the pre-trained linear layers in Llama can be directly projected onto SS1 layers without any finetuning while maintaining model performance. This is a significant finding, as it highlights the effectiveness of our SS1 compression method in preserving the knowledge captured by the pre-trained model. By avoiding the need for finetuning, our approach offers substantial computational savings and facilitates more efficient deployment of compressed models. To validate our approach, we project the transformer layers of the Llama3-8B model onto SS1 layers. Specifically, we use the 'meta-llama/Meta-Llama-3-8b' model provided by

the Hugging Face Transformers library. We then evaluate the compressed model on the MMLU (Massive Multitask Language Understanding) [38] and Winogrande [39] tasks without any further training or finetuning using Language Model Evaluation Harness [40].

For calibration purposes, we utilize two small datasets: tinyMMLU and tinyWinogrande [41]. These datasets consist of 100 curated examples from the original MMLU and Winogrande datasets, respectively. By using these tiny datasets, we aim to identify the layers that are less sensitive to compression while minimizing the computational overhead of the calibration process. The results of our experiments are presented in Table 9. We achieve a 25% compression ratio of the decoder layers in Llama3-8B without any finetuning. This compression is obtained by projecting layers 18, 19,20, 21, 22, 23, 24, 25, 26, 27 in the case of MMLU and layers 18, 19, 20, 21, 22, 23, 24, 25, 26, 30 in the case of Winogrande onto SS1 at 8x compression, resulting in a net compression ratio of approximately 0.25. The selection of layers for compression is based on their sensitivity to compression, as determined by the performance on the tinyMMLU and tinyWinogrande datasets. The layers chosen using the tiny calibration datasets generalize well to the full MMLU and Winogrande datasets. This observation suggests that the compressed model retains the essential knowledge captured by the pre-trained Llama model, enabling it to perform competitively on the complete datasets without any finetuning.

Table 9 presents the evaluation metrics of the compressed Llama model on the full MMLU and Winogrande datasets. On MMLU, our compressed model achieves an accuracy of 61.26%, which is only a slight decrease from the 65.05% accuracy of the original Llama3-8B model. Similarly, on Winogrande, the compressed model obtains an accuracy of 69.93%, compared to the 76.1% accuracy of the uncompressed model. These results demonstrate the effectiveness of our SS1 compression method in maintaining the performance of the Llama model while significantly reducing its parameter count. Furthermore, the compressed Llama model exhibits improved inference speed, achieving up to 1.1x increase in throughput compared to the original model.

The ability to compress large language models like Llama without finetuning opens up new possibilities for efficient deployment and utilization of these models in various applications. Our SS1 compression method offers a practical solution for reducing the computational and memory requirements of large models while preserving their performance on downstream tasks.

Table 13: Llama Projection Results

| Model | #param | MMLU | Wingograte | Speedup |
|---|---|---|---|---|
| LLAMA-3-8b | 8.03B | $65.05 \pm 4e\text{-}3$ | $76.1 \pm 1e\text{-}2$ | 1x |
| SS1 | 6.12B | $61.26 \pm 4e\text{-}3$ | $69.93 \pm 1e\text{-}2$ | 1.1x |

## I.4 Vision

We use the following MLPMixer models We use the repository: https://github.com/omihub777/MLP-

| | MLPMixer-S | MLPMixer-M | MLPMixer-L |
|---|---|---|---|
| #layers | 8 | 8 | 8 |
| hidden-size | 128 | 512 | 1024 |
| hidden-c | 512 | 2048 | 4096 |
| hidden-s | 64 | 256 | 512 |

Mixer-CIFAR with default settings for all runs. We do not use hyperparameter tuning for compression methods.

